# Deep Homomorphism Networks

**Takanori Maehara**[*]
Roku, Inc.
Cambridge, UK
tmaehara@roku.com

**Hoang NT**
University of Tokyo
Tokyo, Japan
hoangnt@g.ecc.u-tokyo.ac.jp

## Abstract

Many real-world graphs are large and have some characteristic subgraph patterns, such as triangles in social networks, cliques in web graphs, and cycles in molecular networks. Detecting such subgraph patterns is important in many applications; therefore, establishing graph neural networks (GNNs) that can detect such patterns and run fast on large graphs is demanding. In this study, we propose a new GNN layer, named *graph homomorphism layer*. It enumerates local subgraph patterns that match the predefined set of patterns $\mathcal{P}^{\bullet}$, applies non-linear transformations to node features, and aggregates them along with the patterns. By stacking these layers, we obtain a deep GNN model called *deep homomorphism network (DHN)*. The expressive power of the DHN is completely characterised by the set of patterns generated from $\mathcal{P}^{\bullet}$ by graph-theoretic operations; hence, it serves as a useful theoretical tool to analyse the expressive power of many GNN models. Furthermore, the model runs in the same time complexity as the graph homomorphisms, which is fast in many real-word graphs. Thus, it serves as a practical and lightweight model that solves difficult problems using domain knowledge.

## 1  Introduction

### 1.1  Background

*Graph neural network (GNN)* is a type of neural network that takes a graph as input. It has been applied to many problems in various domains, such as influence prediction in social networks [60], page ranking in web graphs [65], and chemical prediction in biological networks [39]. See textbooks [46, 32, 71] for the basics of GNN.

The expressive power of GNNs is the central research topic in GNN [63, 75]. A recent interest in this topic is the detectability of subgraph patterns. Many graphs that appear in practice have typical subgraph patterns. For example, social networks have many triangles, which indicates the clustering structure of the society. Web graphs have many cliques that represent clusters of websites, such as link farms. Molecular networks have benzene structures. Since detecting these subgraph patterns is a common strategy in network science [52] and graph data mining [15], we expect that GNNs applied in these fields equip expressive power to detect such patterns. Furthermore, since the graphs in these applications are typically large, we also expect that the GNNs applied in these fields run fast.

Unfortunately, most of the existing GNN models do not meet these expectations. The commonly used GNNs, called *message-passing GNNs (MPGNNs)*, do not meet the expectation of expressive power, as they can only detect tree-shaped patterns [72, 19]. More complex GNNs can detect subgraph patterns, but typically do not meet either expectation: *Higher-order GNNs* assign values to $k$-tuples of nodes instead of nodes [53, 50, 36]. They have the same expressive power as the $k$-dimensional

---

[*]Authors are listed in alphabetical order.

Weisfeiler–Lehman ($k$-WL) test[1], which detects subgraphs of treewidth at most $k$ [23]; however, their complexity is typically $\Omega(n^k)$, which is not applicable to large graphs. *Subgraph GNNs* take a small subgraph for each node and apply a GNN to compute an embedding [77]. Its expressive power depends on the choice of the subgraph selection policy and the base GNN, and the standard choice of the policy and the base GNN, it is strictly more expressive than the 1-WL test but less expressive than the 2-WL or 3-WL tests [77, 25], which is often insufficient to capture the patterns of interest.

One promising direction is *explicit pattern detection*, which explicitly scans the patterns in the graph and uses that information. This approach has been studied and applied in practice for a long time before the GNN era [15, 52, 69, 26], and recent studies combine them with GNN [47, 56, 4, 9, 78, 49, 57]. This approach requires domain knowledge (or "subgraph feature engineering") of what patterns will be important, but often provides a more effective and efficient solution.

Amongst multiple notions of pattern enumeration, here we focus on *graph homomorphisms*, which is the adjacency-preserving mappings from a pattern to the target graph (see Section 2.2 for the definition). We focus on the following two theoretical GNN studies based on graph homomorphisms. The first is by NT and Maehara [56], who extended the homomorphism number to graphs with features and proposed using them as features of downstream models such as support vector machines. The limitation of this approach is that it is inefficient in achieving a higher expressive power — Their approach specifies a set of patterns $\mathcal{P}$ and computes the generalised homomorphism number for each $P \in \mathcal{P}$. This detects all $P \in \mathcal{P}$ (finite number of patterns) using $\Omega(|\mathcal{P}|)$ time. On the other hand, MPGNNs such as GIN detect all $T \in \mathcal{T}$, where $\mathcal{T}$ is the set of trees (infinite set of patterns) without incurring a time complexity of $\Omega(|\mathcal{T}|)$. The second is by Barceló et al. [4], who proposed to add the precomputed rooted homomorphism numbers from the specified patterns $\mathcal{P}^\bullet$ as node features of the graph and to apply MPGNN. The important finding is that such a simple approach boosts that the model detects all $F \in \mathcal{F}^\bullet$ where where $\mathcal{F}^\bullet$ is a set of graphs obtained by attaching a pattern $P \in \mathcal{P}^\bullet$ to nodes of a tree (called $\mathcal{P}^\bullet$-trees) while keeping the time complexity of $O(|\mathcal{P}^\bullet|)$ instead of $O(|\mathcal{F}^\bullet|)$. This approach cannot capture the features of the patterns, which are important in many GNN applications, and the patterns it captures are limited.

Our goal is to establish a connection with the GNN architecture and homomorphisms by extending this line of studies. We observe that, according to the proof of [4], the method of [56] is inefficient in achieving a higher expressive power because it is not "deep" (see Remark 5.4). Therefore, our strategy to achieve more expressive GNNs is to deeply stack homomorphism-based layers.

## 1.2 Our Contribution

In this study, we propose *generalised rooted graph homomorphism number*, which applies a non-linear transform to node features and then aggregates them along with graph homomorphisms (Section 3). We then propose *graph homomorphism layer* that computes the generalised rooted homomorphism numbers with learnable non-linear transforms from a set of patterns $\mathcal{P}^\bullet$ specified as a hyperparameter. We refer to a GNN that stacks this layer *deep homomorphism network* (DHN or $\mathcal{P}^\bullet$-DHN to clarify the patterns). See Figure 1 for an illustration and Section 4 for the details of our model.

By construction, our layer is trained and evaluated in the same time complexity as the generalised rooted homomorphism numbers. Computing the homomorphism number is W[1]-hard in general [17]; however, in many practical cases, such as bounded degree graphs and bounded degeneracy graphs, we can obtain faster algorithms by using the technique in graph homomorphisms (Section 4.2).

The expressive power of the model is analysed using a methodology similar to that in [4]. Let $\overline{\mathcal{P}^\bullet}$ be a set of graphs obtained by iteratively attaching $P^\bullet \in \mathcal{P}^\bullet$ to the singleton (e.g., the set of trees is obtained from an edge by this construction). Then, we can show that the expressive power of $\mathcal{P}^\bullet$-DHN is characterised by the $\mathcal{F}^\bullet$-homomorphism distinguishability (Theorem 5.2). This characterisation is useful for establishing the expressive power hierarchy of the GNN models. In particular, we can discuss its relationship with $k$-GNN and subgraph GNNs using the underlying homomorphism patterns. Another important consequence of this theorem is that it reveals the advantage of stacking multiple GNN layers. Simply put, adding one layer corresponds to adding base patterns to each node in the current set of patterns. Hence, GNN can detect exponentially many patterns and linearly deeper patterns with respect to the number of GNN layers (Remark 5.4 in Section 5.1).

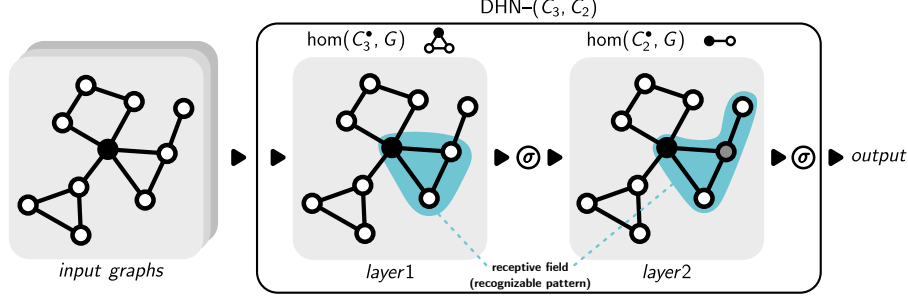

Figure 1: Example Deep Homomorphism Network (DHN) built from two $\mathcal{P}^\bullet$-homomorphism layers: $C_3^\bullet$ and $C_2^\bullet$. By stacking different homomorphism patterns, DHN can detect new patterns without explicit specifications. This figure demonstrates that the "spoon" pattern can be detected by stacking $C_3$ and $C_2$ homomorphism layers.

Essentially, $\mathcal{P}^\bullet$-DHN is a "deep" version of NT and Maehara [56]'s method. Barceló et al. [4]'s method is a DHN that uses the $\mathcal{P}^\bullet$-homomorphism layer for the first layer and the MPGNN layer for the subsequent layers. This generalisation elucidates the relationship between the GNN architecture and the corresponding homomorphisms, thereby facilitating a better understanding of the expressive power hierarchy among different GNN architectures (Section 5.3).

The DHN model takes advantage of pattern enumeration and deep learning. Hence, we expect the model to solve difficult graph problems that require the capture of subgraph patterns at reasonable computational costs. We conducted experiments and observed that the DHN solved difficult benchmark problems (CSL, EXP, and SR25) with fewer parameters than the existing models. For real-world datasets, the proposed model showed promising results, but was still not competitive to the state-of-the-art models that involve a lot of engineering (see Section 6 for discussion).

## 2 Preliminaries

### 2.1 Graphs

A *graph* $G = (V(G), E(G))$ is a pair of *nodes* $V(G)$ and *edges* $E(G)$. We denote by $e = (u, v)$ an edge between $u$ and $v$ [2] and $N(u) = \{v : (u, v) \in E(G)\}$ the neighbours of $u$. An *isomorphism from $G_1$ to $G_2$* is a bijection $\pi \colon V(G_1) \to V(G_2)$ such that $(u_1, v_1) \in E(G_1)$ if and only if $(\pi(u_1), \pi(v_1)) \in E(G_2)$. Two graphs $G_1$ and $G_2$ are *isomorphic* if there exists an isomorphism.

We fix a compact set $\mathcal{X} \subseteq \mathbb{R}^{d_{\text{in}}}$ for the feature space. A *graph with features* is a pair $(G, x)$ of a graph $G$ and a collection $[x_u \in \mathcal{X} : u \in V(G)] \subseteq \mathcal{X}^{V(G)}$ of node features. Two graphs with features, $(G_1, x_1)$ and $(G_2, x_2)$, are *isomorphic* if there is an isomorphism $\pi$ from $G_1$ to $G_2$ such that $x_{1,u_1} = x_{2,\pi(u_1)}$ for all $u_1 \in V(G_1)$.

In this study, we mainly consider the node classification as it is a building block of all other GNN applications. We employ *rooted graph formulation* [47, 56]. A *rooted graph* $G^r$ is a graph $G = (V(G), E(G))$ with a distinguished node $r \in V(G)$. We denote by $G^\bullet$ if there is no need to specify the name of the root node. Two rooted graphs $G_1^{r_1}$ and $G_2^{r_2}$ are *isomorphic* if there is an isomorphism $\pi$ from $G_1$ to $G_2$ such that $r_2 = \pi(r_1)$. The isomorphism of rooted graphs with features is defined similarly. A function $f$ that takes a rooted graph with features $(G, x)$ and produces some quantity is said to be *equivariant* if $f((G_1^\bullet, x_1)) = f((G_2^\bullet, x_2))$ if $(G_1^\bullet, x_1)$ and $(G_2^\bullet, x_2)$ are isomorphic. This study only considers equivariant functions because it is a natural and desirable property for the task (otherwise, the output depends on a synthetic ordering of nodes). Note that if we drop the equivariance, it is easy to construct arbitrary expressive models [54, 64, 43, 18].

*Remark* 2.1 (Advantage of Rooted Graph Formulation). Many existing studies formulate a node classification function as a function that takes a graph $G$ as input and produces $\mathbb{R}^{V(G) \times d}$ matrix as an output. Therefore, mathematically, its codomain is the disjoint union $\bigcup_{G \in \mathcal{G}} \mathbb{R}^{V(G) \times d}$ where $\mathcal{G}$ is

the set of all graphs. Existing studies mitigated such a complex codomain by assuming that all $G$ share the same node set, $V(G) = \{1, \ldots, n\}$, but this creates a limitation on the number of nodes. The rooted graph formulation has no such issue as the domain is the set of rooted graphs and the codomain is $\mathbb{R}^d$. Note that the statement for rooted graphs is easily converted to non-rooted graphs.

## 2.2 Graph Homomorphism

A *graph homomorphism* from a graph $F$ to a graph $G$ is a mapping $\pi \colon V(F) \to V(G)$ such that $(i, j) \in E(F)$ implies $(\pi(i), \pi(j)) \in E(G)$; we refer to $F$ as *pattern graph* and $G$ as *host graph*. As each homomorphism defines a subgraph of $G$ as a homomorphism image $\pi(F) \subseteq G$, we can recognise that a homomorphism represents a $F$-pattern in $G$.

We denote by $\mathrm{Hom}(F, G)$ the set of graph homomorphisms from $F$ to $G$ and $\hom(F, G)$ by its cardinality, called *graph homomorphism number*. If we know $\hom(F, G)$ for multiple $F$, we can obtain a lot of information on the structure of $G$. For example, $\hom(C, G_1) = \hom(C, G_2)$ for all cycles $C$ means that $G_1$ and $G_2$ are cospectral and $\hom(F, G_1) = \hom(F, G_2)$ for all graphs $F$ means that $G_1$ and $G_2$ are isomorphic [44]. See Hell and Nesetril [33] for the basics of graph homomorphisms.

For rooted graphs $F^r$ and $G^s$, a *rooted graph homomorphism*[3] is a homomorphism from $F$ to $G$ that maps $r$ to $s$. We denote by $\mathrm{Hom}(F^r, G^s)$ the set of rooted homomorphisms from $F^s$ to $G^s$.

## 2.3 Weisfeiler-Lehman Test

The *(one-dimensional) Weisfeiler-Lehman test (WL test or 1-WL test)* is a procedure to identify whether given two graphs (with features) are non-isomorphic or potentially isomorphic [30]. The WL-test calculates the "colour $c_u$" of nodes $u$ using the following recursive procedure:

$$c_u^{(0)} = x_u, \qquad c_u^{(k+1)} = \left( c_u^{(k)}, \left\{\!\!\left\{ c_v^{(k)} : v \in N(u) \right\}\!\!\right\} \right), \tag{1}$$

where $\{\!\!\{\}\!\!\}$ denotes the multiset. Here, each colour is a nested tuple of vectors and multisets; a practical implementation applies a hash function to them, but they are equivalent in theory. Let $c^{(k)}(G) = \left\{\!\!\left\{ c_u^{(k)} : u \in V(G) \right\}\!\!\right\}$ be the multiset of colours in the $k$-th step. If $c^{(k)}(G_1) \neq c^{(k)}(G_2)$ for some $k$, then $G_1$ and $G_2$ are not isomorphic. Dvořák [23] proved that two graphs $G_1$ and $G_2$ are indistinguishable by the WL-test if and only if $\hom(T, G_1) = \hom(T, G_2)$ for all trees $T$.

## 2.4 Graph Neural Networks

*Graph neural network (GNN)* is a neural network that takes a graph as input. The most commonly used GNN is a *message-passing GNN* (MPGNN), which computes the node values by

$$h_u^{(0)} = \rho^{(0)}(x_u), \qquad h_u^{(k+1)} = \rho^{(k+1)} \left( h_u^{(k)}, \phi^{(k)} \left( \left\{\!\!\left\{ h_v^{(k)} : v \in N(u) \right\}\!\!\right\} \right) \right), \tag{2}$$

where $\rho^{(k)}$ is a learnable function and $\phi^{(k)}$ is a (learnable) multi-set function, i.e., a permutation-invariant function for the arguments, for each $k$. It is easy to see that the MPGNN defines equivariant functions. A typical implementation of MPGNN is *graph isomorphism network (GIN)* [72], which uses the summation for $\phi^{(k)}$.

Due to the similarity between the WL test (1) and the MPGNN (2), it can be proved that the expressive power of the MPGNN is identical to the WL test [72]. As the WL-indistinguishability is equivalent to the homomorphism-indistinguishability from all trees, as mentioned above, we can conclude that MPGNN can only detect tree-shaped patterns.

## 3 Generalised Homomorphism Numbers for Rooted Graphs with Features

A *pattern graph with transformations* is a pair $(F^\bullet, \mu)$ of a rooted graph $F^\bullet$ and a collection of continuous functions $\mu = \{\mu_p : p \in V(F^\bullet)\}$ defined on the nodes of $F^\bullet$, where each $\mu_p$ maps their

inputs to $\mathbb{R}^d$. The *generalised rooted homomorphism number* $\mathrm{hom}((F^\bullet, \mu), (G^\bullet, x))$ from a pattern graph with transformations $(F^\bullet, \mu)$ to a rooted graph with features $(G^\bullet, x)$ is then defined by

$$\mathrm{hom}((F^\bullet, \mu), (G^\bullet, x)) := \sum_{\pi \in \mathrm{Hom}(F^\bullet, G^\bullet)} \prod_{p \in V(F^\bullet)} \mu_p(x_{\pi(p)}), \tag{3}$$

where the product in the right-hand side is the element-wise product. Note that NT and Maehara [56]'s generalised homomorphism is our special case, which uses the same transformation to all nodes.

A generalised homomorphism number maps a graph with features to a real vector (not necessarily a number). By definition, two isomorphic graphs with features have the same generalised homomorphism numbers for any pattern graph with transformations. Here, the converse also holds.

**Theorem 3.1.** *Let $(G_1^\bullet, x_1)$ and $(G_2^\bullet, x_2)$ be rooted graphs with features. $(G_1^\bullet, x_1)$ and $(G_2^\bullet, x_2)$ are isomorphic if and only if $\mathrm{hom}((F^\bullet, \mu), (G_1^\bullet, x_1)) = \mathrm{hom}((F^\bullet, \mu), (G_2^\bullet, x_2))$ for any pattern graphs with transformations $(F^\bullet, \mu)$.*

*Proof Sketch.* We use the Lovasz theorem that any finite relational structure is determined from the number of homomorphisms [44]. First, we recognise graphs with features as a relational structure consists of the adjacency relation and feature value relation. Then, we show that the number of homomorphisms as the relational structure (i.e., the number of mappings that preserve the edges and feature values) is computed using our generalised homomorphism by suitably choosing $\mu$. □

Let $\mathcal{F}^\bullet$ be a set of pattern graphs with transformations. We say that two rooted graphs with features, $(G_1^\bullet, x_1)$ and $(G_2^\bullet, x_2)$, are $\mathcal{F}^\bullet$-*homomorphism indistinguishable* if $\mathrm{hom}((F^\bullet, \mu), (G_1^\bullet, x_1)) = \mathrm{hom}((F^\bullet, \mu), (G_2^\bullet, x_2))$ for all $(F^\bullet, \mu) \in \mathcal{F}^\bullet$; Theorem 3.1 states that $\mathcal{F}^*$-homomorphism indistinguishability coincides with the isomorphism if $\mathcal{F}^*$ is the set of all pattern graphs with transformations. In general, homomorphism indistinguishability forms an equivalence relation.

*Remark* 3.2. In graph homomorphism literature, *weighted homomorphism number* [45] is studied more frequently. It is essentially a generalised homomorphism number with linear transformations, and it cannot distinguish some non-isomorphic graphs [12, 70]. However, as shown in the above, our generalised homomorphism mitigates this issue by introducing the non-linearity of $\mu$.

## 4 Deep Homomorphism Networks

### 4.1 Definition

Let $\mathcal{P}^\bullet$ be a set of rooted graphs. A *graph homomorphism layer with respect to* $\mathcal{P}^\bullet$ is a GNN layer defined using the generalised homomorphism number as follows:

$$\mathrm{GHL}_{\mathcal{P}^\bullet}((G^u, x); \rho, \{\mu_{P^\bullet} : P^\bullet \in \mathcal{P}^\bullet\}) = \rho\left(\mathrm{hom}((P^\bullet, \mu_{P^\bullet}), (G^u, x)) : P^\bullet \in \mathcal{P}^\bullet\right), \tag{4}$$

where $(G^u, x)$ is the input rooted graph with features, and $\rho$ and $\mu_{P^\bullet, p}$ for all $P^\bullet \in \mathcal{P}^\bullet$ and $p \in V(P^\bullet)$ are neural networks. We often omit neural network parameters and write it as $\mathrm{GHL}_{\mathcal{P}^\bullet}((G^u, x))$. The input dimensionality of $\rho$ is the sum of the output dimensionalities of $\mu_{P^\bullet, u}$, and the input dimensionality of $\mu_{P^\bullet, u}$ is the dimensionality of the input $h$. The layer defines an equivariant function since the graph homomorphism numbers are equivariant functions.

*Deep homomorphism network (DHN)* is a neural network obtained by "deeply" stacking the graph homomorphism layers as follows:

$$h^{(0)} = x, \qquad h^{(k+1)} = \mathrm{GHL}_{\mathcal{P}^{(k)\bullet}}((G^u, h^{(k)})). \tag{5}$$

We denote by $(\mathcal{P}^{(1)\bullet}, \mathcal{P}^{(2)\bullet}, \dots)$-DHN if we want to emphasize the pattern sets, and we denote $\mathcal{P}^\bullet$-DHN for $(\mathcal{P}^\bullet, \mathcal{P}^\bullet, \dots)$-DHN. By definition, a DHN is an equivariant function. The number of parameters in DHN is proportional to the number of nodes in the pattern graphs.

**Example 4.1** (DHN generalises MPGNN). Let $\mathcal{P}^\bullet = \{\bullet, \bullet - \circ\}$ be the patterns consisting of single-node and single-edge graphs. Here, we see that the $\mathcal{P}^\bullet$-DHN is a MPGNN.

We first consider the single-node graph $\bullet$. There is the unique homomorphism from $\bullet$ to $G^u$ given by $\pi(\bullet) = u$; hence,

$$\mathrm{hom}((\bullet, \{\mu_{\bullet, \bullet}\}), (G^u, x)) = \mu_{\bullet, \bullet}(x_u). \tag{6}$$

We then consider the single-edge graph $\bullet - \circ$. As the set of homomorphisms from $\bullet - \circ$ to $G^u$ corresponds to the set of edges incident to $u$, we have

$$\mathrm{hom}((\bullet - \circ, \{\mu_{\bullet - \circ, \bullet}, \mu_{\bullet - \circ, \circ}\}), (G^u, x)) = \sum_{v \in N(u)} \mu_{\bullet - \circ, \bullet}(x_u) \mu_{\bullet - \circ, \circ}(x_v). \tag{7}$$

By setting $\mu_{\bullet, \bullet}(x) = x$, $\mu_{\bullet - \circ, \bullet}(x) = 1$, and $\mu_{\bullet - \circ, \circ}(x) = x$ for some $\mu_\circ$, we obtain the MPGNN:

$$\mathrm{GHL}_{\mathcal{P}^\bullet}((G^u, x)) = \rho \left( x_u, \sum_{v \in N(u)} x_v \right). \tag{8}$$

DHN generalises several existing models. We review such results in Sections 5.2.

## 4.2 Computational Complexity

Evaluating a graph homomorphism layer with respect to $\mathcal{P}^\bullet$ on $(G, x)$ takes the same time complexity as evaluating $\mathrm{hom}((P^\bullet, \mu), (G^u, x))$ for all $P^\bullet \in \mathcal{P}^\bullet$ and $u \in V(G)$; therefore, its computational complexity is at least that of $\mathrm{hom}(P, G)$ for some $P^\bullet \in \mathcal{P}^\bullet$. We cannot expect a linear-time algorithm to evaluate this quantity without any assumption because computing $\mathrm{hom}(P, G)$ is a $W[1]$-hard problem parameterised by $|V(P)|$ [29]. However, there are several cases that admit efficient algorithms for computing $\mathrm{hom}(P, G)$. We see that these results can be generalised to our generalised homomorphism numbers as follows.

**Case 1: $P$ has a bounded treewidth**   Treewidth is a parameter that represents how far the graph is from being a tree; see [21] about treewidth. If $P$ has a bounded treewidth, we can compute $\mathrm{hom}(P, G)$ in $O(n^{\mathrm{tw}(P)+1})$ time using the dynamic programming algorithm [20]. The algorithm is easily extended to generalised rooted graph homomorphism numbers; see Section A.1. Hence, we can evaluate the graph homomorphism number in polynomial time in this situation.

**Case 2: $G$ has a bounded degree**   In some examples, such as molecular networks, the host graph $G$ has a small maximum degree. In this case, we can enumerate $\mathrm{Hom}(P^\bullet, G^u)$ in constant time by brute-force enumeration. Therefore, we can evaluate the graph homomorphism layer in linear time.

**Case 3: $G$ has a bounded degeneracy and $P$ has a bounded DAG-treewidth**   A graph $G$ has degeneracy at most $k$ if there is an ordering of nodes $u_1, \ldots, u_n$ such that $|\{j : u_j \in N(u_i), j \geqslant i\}| \leqslant k$ for all $i = 1, \ldots, n$ [41]. Many real-world graphs have small degeneracy [7]. Hence, it is practically important to have algorithms that run fast on graphs of bounded degeneracy. Bressan [10] introduced *DAG-treewidth*, and proposed an algorithm for computing the homomorphisms number in $O(n^{\mathrm{dagtw}(P)})$ time using the dynamic programming algorithm. An important special case is that $P$ has no induced cycles of length greater than five. In this case, the DAG treewidth is one [58] (the converse is also true); hence, we can evaluate the homomorphism numbers in linear time. To clarify the procedure, we put a linear-time algorithm for the quadrangle $C_4$; see Section A.2 in Appendix.

# 5 Theoretical Analysis of $\mathcal{P}^\bullet$-DHN Model

## 5.1 Expressive Power of $\mathcal{P}^\bullet$-DHN Model

In Example 4.1, we observed that a DHN with simple patterns $\{\bullet, \bullet - \circ\}$ contains a MPGNN. Here, we focus to the phenomenon that, although it aggregates local information along with such simple patterns, it has a great expressive power specified as the 1-WL test [72], which distinguishes all $\mathcal{T}^\bullet$ homomorphism-distinguishable graphs, where $\mathcal{T}^\bullet$ is the set of all trees [23]. The goal of this section is to generalise this relation to arbitrary patterns.

Let $F^\bullet$ and $P^r$ be rooted graphs. The *rooted product* of $F^\bullet$ and $P^r$ at $p \in V(F^\bullet)$ is the rooted graph obtained by attaching $r$ at $p$, i.e., $F^\bullet *_p P^r := F^\bullet \cup P^r / \{p, r\}$ [28]; see Figure 2 for an example. Let $\mathcal{F}^\bullet$ and $\mathcal{P}^\bullet$ be sets of rooted graphs. We denote by $\mathcal{F}^\bullet * \mathcal{P}^\bullet = \{F^\bullet *_u P^\bullet : u \in V(F^\bullet)\}$ the set of all rooted products. We denote by $\overline{\mathcal{P}^\bullet} = \bigcup_{l=0,1,\ldots} (\mathcal{P}^\bullet)^{*l}$ the set of all graphs obtained by the iterated rooted products, where $(\mathcal{P}^\bullet)^{*0} = \{\bullet\}$ and $(\mathcal{P}^\bullet)^{*l} = \mathcal{P}^\bullet \cdots * \mathcal{P}^\bullet$ ($l$ times). We can easily verify the following example.

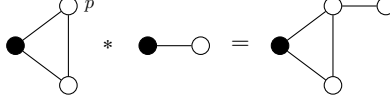

Figure 2: Rooted product of two graphs, the triangle and the edge, at $p$.

**Example 5.1.** $\overline{\{\bullet - \circ\}}$ is the set of all rooted trees $\mathcal{T}^\bullet$.

Now Example 4.1 and Example 5.1 lead to the conjecture that the expressive power of $\mathcal{P}^\bullet$-DHN is characterised by the iterated rooted product $\overline{\mathcal{P}^\bullet}$ of the pattern graph. We prove this as follows, which is the main theorem in this paper.

**Theorem 5.2.** *Let $\mathcal{P}^\bullet$ be a set of rooted graphs. For any two rooted graphs with features $(G_1^\bullet, x_1)$ and $(G_2^\bullet, x_2)$, the following are equivalent.*

    *1. For any $\mathcal{P}^\bullet$-DHN $h$, we have $h(G_1^\bullet, x_1) = h(G_2^\bullet, x_2)$.*

    *2. $(G_1^\bullet, x_1)$ and $(G_2^\bullet, x_2)$ are $\overline{\mathrm{P}^\bullet}$-homomorphism indistinguishable.*

The key lemma to prove this theorem is the following lemma, which decomposes the homomorphism from rooted product into the homomorphisms from the factors.

**Lemma 5.3** (Chain Rule). *Let $F^\bullet$ be a rooted graph obtained by taking the rooted product of $P^\bullet$ and $F_p^\bullet$ at node $p$ for each $p \in V(P^\bullet)$. Then, for any $\mu$, there exists $\mu_p$ such that*

$$\mathrm{hom}((F^\bullet, \mu), (G^\bullet, x)) = \sum_{\pi \in \mathrm{Hom}(P^\bullet, G^\bullet)} \prod_{p \in V(P)} \mathrm{hom}((F_p^\bullet, \mu_p), (G^{\pi(p)}, x)). \qquad (9)$$

*for any rooted graph with features $(G^\bullet, x)$.*

*Proof Sketch of Theorem 5.2.* Instead of proving the equivalence between 1 and 2, we introduce a variant of WL-test, named $\mathcal{P}^\bullet$-*WL test*, and introduce the third statement: $(G_1^\bullet, x_1)$ and $(G_2^\bullet, x_2)$ are $\mathcal{P}^\bullet$-WL indistinguishable, and prove the equivalence of 1, 2, and 3. Here, $3 \Rightarrow 2$ is clear from the definition of the $\mathcal{P}^\bullet$-WL test, which is similar to that of [72]. $1 \Rightarrow 2$ is straightforward by seeing that a generalised homomorphism from any $F^\bullet \in \overline{\mathcal{P}^\bullet}$ is expressed by a DHN. To prove $2 \Rightarrow 3$, we prove that the colour assigned by $\mathcal{P}^\bullet$-WL test is uniquely identified by evaluating suitably-chosen pattern graphs $F^\bullet \in \overline{\mathcal{P}^\bullet}$ with transformations. This part is similar to [4] but we use the chain rule above and a basic results from multi-symmetric polynomials. $\qquad \square$

*Remark* 5.4. Establishing deeper GNN models is a central challenge in GNN community [42]. Although deeper models do not necessarily perform well in *practice* [37, 62], in *theory*, Theorem 5.2 and its proof clearly show the advantage of deeper GNNs in terms of the number of pattern graphs — From the proof of Theorem 5.2, we see that $l$-layer DHN models can count homomorphisms from $2^{O(l)}$ different patterns. In this sense, one could say that "the expressive power of a GNN grows exponentially in the number of layers."

## 5.2 Relationship with Existing Models

In this section, we review the relationship between the proposed DHN and some existing models.

**Example 5.5** (DHN generalises NT and Maehara [56]). Our first-motivated paper, NT and Maehara [56], proposed to compute (their version of) generalised homomorphism number and use it as a feature of downstream models for graph classification. By definition, our DHN can be seen as a multi-layer version of their approach.

**Example 5.6** (DHN generalises Barceló et al. [4]). Our second motivated paper, Barceló et al. [4], proposed to append homomorphism numbers from arbitrary pattern $\mathcal{P}^\bullet$ as node features. This can be seen as a DHN that uses an arbitrary pattern $\mathcal{P}^\bullet$ in the first layer and the MPGNN pattern $\{\bullet, \bullet - \circ\}$ in the subsequent layers, i.e., it is the $(\mathcal{P}^\bullet, \{\bullet, \bullet - \circ\}, \{\bullet, \bullet - \circ\}, \dots)$-DHN. They showed that their model can detect graphs called $\mathcal{P}^\bullet$-patterns, which is obtained by attaching $\mathcal{P}^\bullet$ to nodes of a tree. This follows from our theorem, as the $\mathcal{P}^\bullet$-patterns are exactly the graphs obtained by the rooted product to a tree and $\mathcal{P}^\bullet$.

**Example 5.7** (DHN generalises Paolino et al. [57]). Recently, Paolino et al. [57] proposed a GNN that aggregates information over cycles. Their model is a DHN that uses the set of cycles $\mathcal{C}_l^{\bullet} = \{C_1^{\bullet}, \ldots, C_l^{\bullet}\}$ of lengths at most $l$ as a pattern set, i.e., it is the $\mathcal{C}_l^{\bullet}$-DHN. They showed that their model can detect cactus graphs with a maximum cycle length of $l$. This follows from our theorem since $\overline{\mathcal{C}_l^{\bullet}}$ are the set of such cactus graphs.

**Example 5.8** (DHN generalises the most expressive subgraph GNNs). For each layer, a subgraph GNN takes the $l$-hop neighbours and applies a GNN to compute the value of the root node [77]. The most expressive GNN in this class uses the universal GNN on the subgraph. If the underlying graphs have a bounded degree, such a GNN is an instance of DHN — Let $\mathcal{G}_{d,l}^{\bullet}$ be the set of all rooted graphs of degree at most $d$ and radius at most $l$. Then, the $\mathcal{G}_{d,l}^{\bullet}$ homomorphisms identify $\mathcal{G}_{d,l}^{\bullet}$ [44]. Therefore, $\mathcal{G}_{d,l}^{\bullet}$-DHN has the same expressive power as the subgraph GNN with universal GNN if the underlying graphs have degree at most $d$.

## 5.3 Applications: Expressive Power Hierarchy

Theorem 5.2 is a powerful tool for comparing the expressive power of different GNN models. Let $A$ and $B$ be two GNN models. We say that $A$ is *more expressive than* $B$ (denoted by $A \succeq B$) if $h_A((G_1^{\bullet}, x_1)) = h_A((G_2^{\bullet}, x_2))$ for all $h_A \in A$ implies $h_B((G_1^{\bullet}, x_1)) = h_B((G_2^{\bullet}, x_2))$ for all $h_B \in B$, and $A$ *is strictly more expressive than* $B$ (denoted by $A \succ B$) if the $A \succeq B$ but $B \nsucceq A$. To prove $A \succ B$, we typically show that $A$ can implement $B$, and find a pair of instances $(G_1^{\bullet}, x_1)$, $(G_2^{\bullet}, x_2)$, separating these classes. However, finding such a pair often requires nontrivial work.

Theorem 5.2 reduces the expressive power of $\mathcal{P}^{\bullet}$-DHN model to $\overline{\mathcal{P}^{\bullet}}$-homomorphism indistinguishability, and allows us to use lots of existing work established in graph theory literature [44, 19, 61, 55, 31]. For example, after some preparation (Section D.6), we can easily prove the following hierarchy in a unified way. See also Figure D.6 in Appendix showing hierarchy of some models.

**Corollary 5.9.** *Let $\mathcal{C}_k^{\bullet}$ be a set of cycles of size at most $k$ (where we identify the cycles of length one and two as a singleton and an edge, respectively), $\mathcal{K}_k^{\bullet}$ be the set of cliques of size at most $k$, and $\mathcal{S}_k^{\bullet}$ be the set of connected graphs of size at most $k$. Then, the following holds.*

- *$\mathcal{C}_k^{\bullet}$-DHN model $\precsim \mathcal{C}_{k+1}^{\bullet}$-DHN model, $\mathcal{K}_k^{\bullet}$-DHN model $\precsim \mathcal{K}_{k+1}^{\bullet}$-DHN model, and $\mathcal{S}_k^{\bullet}$-DHN model $\precsim \mathcal{S}_{k+1}^{\bullet}$-DHN model for all $k \geqslant 2$.*

- *$\mathcal{C}_k^{\bullet}$-DHN model $\precsim \mathcal{S}_k^{\bullet}$-DHN model for all $k \geqslant 4$, and $\mathcal{S}_k^{\bullet}$-DHN model is incomparable with $\mathcal{C}_{k+1}^{\bullet}$-DHN model for all $k \geqslant 3$.*

- *$\mathcal{K}_k^{\bullet}$-DHN model $\precsim \mathcal{S}_k^{\bullet}$-DHN model for all $k \geqslant 4$, and $\mathcal{S}_k^{\bullet}$-DHN model is incomparable with $\mathcal{K}_{k+1}^{\bullet}$-DHN model for $k \geqslant 3$*

- *$\mathcal{C}_k^{\bullet}$-DHN model and $\mathcal{K}_k^{\bullet}$-DHN model are incomparable for $k \geqslant 4$.*

We can also prove the relations of expressive powers of existing architectures using our framework as follows. See Section B in the Appendix for a detailed discussion of the existing models.

**Corollary 5.10.** *If $\{\bullet, \bullet - \circ\} \subseteq \mathcal{P}^{\bullet}$ and $\mathcal{P}^{\bullet}$ contains a graph with a cycle, then the $\mathcal{P}^{\bullet}$-DHN model is strictly more expressive than the MPGNN model.*

**Corollary 5.11.** *If the maximum treewidth of $P^{\bullet} \in \mathcal{P}^{\bullet}$ is $k$, then the $\mathcal{P}^{\bullet}$-DHN model is less expressive than the $k$-WL equivalent models such as $(k + 1)$-GNN and $(k + 1)$-IGN models.*

**Corollary 5.12.** *If the maximum chordless cycle length of $P^{\bullet} \in \mathcal{P}^{\bullet}$ is finite, then $\mathcal{P}^{\bullet}$-DHN model is incomparable with 2-WL equivalent models such as 3-GNN and 3-IGN models.*

**Corollary 5.13.** *The subgraph GNN model using the $k$-hop egograph selection policy and universal GNN as a base encoder is strictly more expressive than $\mathcal{S}_k^{\bullet}$-DHN model, and is incomparable with 2-WL models such as 3-GNN and 3-IGN models.*

*Remark* 5.14. Recently, [78] provided homomorphism characterisation of GNN models based on $k$-WL-like tests for $k \geqslant 2$. As all of these models can capture arbitrary long cycles, they are not less expressive than any DHN model.

## 5.4 Continuity and universality of $\mathcal{P}^\bullet$-DHN

One of the desired properties of graph algorithms is the *continuty*. Graphs appear in network science applications are often very large and almost impossible to obtain the full structure. In a such case, we usually sample a smaller graph, conduct analysis, and expect the outcome approximates for the original graph [38]. The continuity guarantees the validity of such a procedure so that the outcomes of the original graph and the sampled graph are close. Such property is referred to as the *size generalisability* in GNN literature [73].

Different sampling procedure introduces different notion of continuity (i.e., topology) in the graph space [48]. Here, we consider the *BFS sampling*, which randomly samples a node, performs $k$-hop breadth-first search (BFS), and select the subgraph induced by the nodes. The topology induced by the BFS sampling is called *Benjamini–Schramm topology* [6, 68]. We claim that $\mathcal{P}^\bullet$-DHN is continuous with respect to this topology.

Formally, we consider the set $\mathcal{G}_d^\bullet$ of rooted graphs with features whose degrees are at most $d$. Let $\mathfrak{G}_d^\bullet$ be the Cauchy completion of $\mathcal{G}_d^\bullet$ with respect to the Benjamini–Schramm distance; see Appendix for the precise definition. Then, we can prove the following.

**Lemma 5.15.** *For any finite $\mathcal{P}^\bullet$, a $\mathcal{P}^\bullet$-DHN is a continuous function on $\mathfrak{G}_d^\bullet$ with respect to the Bejnamini–Schramm topology.*

This lemma has some applications. The first one is the universal approximation. We say that a function $f$ is $\mathcal{F}^\bullet$-*homomorphism indistinguishable* if $f((G_1^\bullet, x_1)) = f((G_2^\bullet, x_2))$ for any $\mathcal{F}^\bullet$-homomorphism indistinguishable $(G_1^\bullet, x_1)$ and $(G_2^\bullet, x_2)$. We can show that any $\overline{\mathcal{P}^\bullet}$-homomprhism indistinguishable function is arbitrary accurately approximated by the DHN model as follows, which guarantees the validity of using $\mathcal{P}^\bullet$-DHN model for tasks that $\overline{\mathcal{P}^\bullet}$ substructure is relevant.

**Theorem 5.16.** *For any integer $d$ and a finite $\mathcal{P}^\bullet$, the $\mathcal{P}^\bullet$-DHN model is dense in the set of all $\overline{\mathcal{P}^\bullet}$-indistinguishable continuous functions on $\mathfrak{G}_d^\bullet$.*

Another application is the comparison with existing GNN models as follows.

**Example 5.17** (DHN is incomparable with Zhang et al. [76]). Recently, Zhang et al. [76] observed that many linear-time GNN models could not detect biconnectivity, and they proposed a new model that can detect biconnectivity. Their observation is true because the biconnectivity is not a continuous property in the Benjamini–Schramm topology, and most linear-time models, including DHN, are continuous in this topology. That is, for any continuous model, there are sufficiently close biconnected graph $G_1$ and non-biconnected graph $G_2$ such that the continuous model fails to detect their difference. Conversely, any model that can detect the biconnectivity must be non-continuous in the Benjamini–Schramm topology. Therefore, such models might not have size-generalisability, which is not suitable for large graph applications.

## 6 Experiments

**Experimental Setting** We present the experimental results on the three most common synthetic benchmark datasets for GNN expressivity and two real-world graph classification datasets. The Circular Skip Links (CSL) dataset consists of 150 undirected regular graphs of degree four [54]. EXP [1] and SR25 [2, 56] are datasets not distinguishable by 1-WL (EXP) and 3-WL (SR25). The ENZYMES [66, 8] and PROTEINS [8, 22] datasets represent the protein function prediction task formulated as the graph classification problem[4] We set the same experimental setting as previous works [1, 34, 24], see the Appendix C for more details of these datasets. For our DHN, we use two sets of patterns as the building blocks. $C_{i:j} = \{C_i, \ldots, C_j\}$ denotes the sets of cycles of lengths $i$ to $j$. Similarly, $K_{i:j} = \{K_i, \ldots, K_j\}$ denotes the set of cliques of size $i$ to $j$. We use 3-layer MLPs for both $\rho$ and $\mu_p$ for the homomorphism layer (Eq. (4)). In Table 1, we present the models' configurations inside the single brackets. For example, DHN–$(C_2K_{3:5}, C_2K_{3:5})$ means the model has two layers, and each layer consists of 4 kernels: $C_2, K_3, K_4,$ and $K_5$. Note that $K_4$ has the treewidth of four; hence, the DHN with $K_4$ is incomparable with PPGN, $I^2$-GNN, and $N^2$-GNN.

**Results** Overall, we see that the performance of DHN depends on the choice of the pattern graphs. For a suitable choice (i.e., the last row), it can solve all the benchmark problems. CSL is easy and

Table 1: Experimental results on synthetic and real-world datasets for GNN expressivity (Acc.%)

| | #params | CSL | EXP | SR25 | ENZYMES | PROTEINS |
|---|---|---|---|---|---|---|
| MPNN (4 layers) [72] | 27k | 0 | 0 | 0 | $54.6 \pm 4.5$ | $72.0 \pm 4.0$ |
| PPGN (4 layers) [51] | 96k | **100** | **100** | 0 | $58.2 \pm 5.7$ | $77.2 \pm 3.7$ |
| $I^2$-GNN (4 layers) [34] | 143k | **100** | **100** | **100** | - | - |
| $N^2$-GNN (4 layers) [24] | 355k | **100** | **100** | **100** | - | - |
| DHN–$(C_{2:4})$ | 5k | **100** | 50 | 0 | $64.3 \pm 5.5$ | $76.5 \pm 3.0$ |
| DHN–$(C_{2:5})$ | 7k | **100** | 81 | 0 | $63.7 \pm 5.4$ | $77.0 \pm 3.2$ |
| DHN–$(C_{2:10})$ | 27k | **100** | 98 | 0 | $58.0 \pm 5.3$ | $78.5 \pm 2.5$ |
| DHN–$(C_2 K_{3:5})$ | 7k | **100** | 50 | 53 | $63.3 \pm 5.5$ | $76.0 \pm 2.7$ |
| DHN–$(C_{2:4}, C_2)$ | 8k | **100** | 50 | 0 | $64.4 \pm 5.9$ | $77.1 \pm 2.8$ |
| DHN–$(C_{2:5}, C_2)$ | 11k | **100** | 99 | 0 | $62.0 \pm 5.5$ | $77.0 \pm 2.5$ |
| DHN–$(C_{2:5}, C_{2:5})$ | 36k | **100** | 99 | 0 | $59.9 \pm 5.2$ | $76.7 \pm 3.3$ |
| DHN–$(C_{5:10}, C_2)$ | 27k | **100** | **100** | 0 | $63.5 \pm 6.1$ | $78.2 \pm 3.3$ |
| DHN–$(C_2 K_{3:5}, C_2 K_{3:5})$ | 36k | **100** | **100** | **100** | $57.5 \pm 6.6$ | $77.4 \pm 3.4$ |

can be solved with any model (except the MPNN, aka. GIN). EXP is not co-spectral; hence, we can detect the difference by using cycles; as shown in the table, using more cycles improves the performance. An important observation here is that stacking layer often boosts the expressive power of the DHN models — the single-layer model DHN–$(C_{2:5})$ can only achieve 81% while adding one extra layer, DHN–$(C_{2:5}, C_2)$ achieves 99% accuracy. The same phenomenon is observed in other models except DHN–$(C_{2:4})$. SR25 is co-spectral; hence, adding cycles does not help solve the problem. Experimentally, we found that adding $K_{3:5}$ solved the problem. Furthermore, stacking layers helped both in training convergence and achieving better results. In general, the DHN models have fewer parameters than the existing highly expressive GNNs because they are designed to capture a limited set of patterns, which leads to fast and low-memory training.[5]

We report the stratified 10-fold cross-validation accuracies for ENZYMES and PROTEINS datasets in Table 1. Our proposed models performed comparably to other much larger high-expressivity models on these real-world datasets. Although our results are still far from the reported state-of-the-art results (78% for ENZYMES and 84% for PROTEINS), we believe DHN has the potential to be improved beyond the theoretical context of this paper.

## 7 Conclusion

In this study, we developed a new GNN named deep homomorphism network (DHN). DHN is parameterised by a set of base patterns $\mathcal{P}^{\bullet}$, which is typically specified by the domain knowledge and computational complexity. The expressive power of the model is completely characterised by the homomorphism numbers from any patterns generated from $\mathcal{P}^{\bullet}$. Moreover, the model is evaluated efficiently in several cases, including the patterns having bounded treewidth, graphs having bounded degree, the patterns having bounded DAG-treewidth, and the graphs having bounded degeneracy.

**Limitation** The DHN model is motivated by network science applications that involve large and sparse graphs. Therefore, it might not be suitable for other applications. More specifically, using DHN might not be competitive in the following situations: (1) when the graphs are small so that $O(n^k)$ time complexity of $k$-WL graph neural networks is acceptable. This is commonly seen in graph classification tasks. (2) when the graphs are dense so that pattern enumeration takes $\Omega(n^k)$ time. Simple models such as MPGNN would be more suitable for such case.

**Future Work** Essentially, our DHN is a "homomorphism extension" of the MPGNN model; therefore, it is fundamentally impossible to capture arbitrary long cycles. A promising future work is to establish the corresponding theory for the local $k$-GNN for $k \geqslant 2$, which allows us to capture arbitrary long cycles attached to small complex patterns which appear in biological networks. Such work will require combining our construction on top of the recently established homomorphism characterisation of local WLs [78].

## Footnotes

[1]There are a few definitions of WL tests with inconsistent dimension counts. We follow [13]'s definition, which is also called the folklore Weisfeiler–Lehman test.

[2]We consider simple undirected graphs, which is the standard setting in the theory of graph homomorphisms. The proposed method can be easily extended to nonsimple directed graphs.

[3]Paul-Pena and Seshadhri [58] called this *vertex homomorphism*.

[4]These datasets are parts of the TUDataset collection.

[5]The source code for DHN is provided at `https://github.com/gear/dhn`.

[6]$\mathcal{B}_k$ is an infinite set of graphs. Thus, we need to consider each degree $d$ independently.

# References

[1] Ralph Abboud, Ismail Ilkan Ceylan, Martin Grohe, and Thomas Lukasiewicz. The surprising power of graph neural networks with random node initialization. In *International Joint Conference on Artificial Intelligence (IJCAI'21)*, 2021.

[2] Muhammet Balcilar, Pierre Héroux, Benoit Gauzere, Pascal Vasseur, Sébastien Adam, and Paul Honeine. Breaking the limits of message passing graph neural networks. In *International Conference on Machine Learning*, pages 599–608. PMLR, 2021.

[3] Albert-László Barabási. The new science of networks. *Cambridge MA. Perseus*, 2002.

[4] Pablo Barceló, Floris Geerts, Juan Reutter, and Maksimilian Ryschkov. Graph neural networks with local graph parameters. *Advances in Neural Information Processing Systems*, 34:25280–25293, 2021.

[5] Paul Beaujean, Florian Sikora, and Florian Yger. Graph homomorphism features: Why not sample? In *Joint European Conference on Machine Learning and Knowledge Discovery in Databases*, pages 216–222. Springer, 2021.

[6] Itai Benjamini and Oded Schramm. Recurrence of distributional limits of finite planar graphs. *Selected Works of Oded Schramm*, pages 533–545, 2011.

[7] Suman K Bera, Amit Chakrabarti, and Prantar Ghosh. Graph coloring via degeneracy in streaming and other space-conscious models. In *47th International Colloquium on Automata, Languages, and Programming (ICALP 2020)*. Schloss-Dagstuhl-Leibniz Zentrum für Informatik, 2020.

[8] Karsten M Borgwardt, Cheng Soon Ong, Stefan Schönauer, SVN Vishwanathan, Alex J Smola, and Hans-Peter Kriegel. Protein function prediction via graph kernels. *Bioinformatics*, 21 (suppl_1):i47–i56, 2005.

[9] Giorgos Bouritsas, Fabrizio Frasca, Stefanos Zafeiriou, and Michael M Bronstein. Improving graph neural network expressivity via subgraph isomorphism counting. *IEEE Transactions on Pattern Analysis and Machine Intelligence*, 45(1):657–668, 2022.

[10] Marco Bressan. Faster subgraph counting in sparse graphs. In *14th International Symposium on Parameterized and Exact Computation (IPEC 2019)*. Schloss-Dagstuhl-Leibniz Zentrum für Informatik, 2019.

[11] Emmanuel Briand. When is the algebra of multisymmetric polynomials generated by the elementary multisymmetric polynomials? *Beiträge zur Algebra und Geometrie: Contributions to Algebra and Geometry, 45 (2), 353-368.*, 2004.

[12] Jin-Yi Cai and Artem Govorov. On a theorem of lovász that (&sdot, h) determines the isomorphism type of h. *ACM Transactions on Computation Theory (TOCT)*, 13(2):1–25, 2021.

[13] Jin-Yi Cai, Martin Fürer, and Neil Immerman. An optimal lower bound on the number of variables for graph identification. *Combinatorica*, 12(4):389–410, 1992.

[14] Norishige Chiba and Takao Nishizeki. Arboricity and subgraph listing algorithms. *SIAM Journal on computing*, 14(1):210–223, 1985.

[15] Diane J Cook and Lawrence B Holder. Graph-based data mining. *IEEE Intelligent Systems and Their Applications*, 15(2):32–41, 2000.

[16] Radu Curticapean, Holger Dell, and Dániel Marx. Homomorphisms are a good basis for counting small subgraphs. In *Proceedings of the 49th Annual ACM SIGACT Symposium on Theory of Computing*, pages 210–223, 2017.

[17] Víctor Dalmau and Peter Jonsson. The complexity of counting homomorphisms seen from the other side. *Theoretical Computer Science*, 329(1-3):315–323, 2004.

[18] George Dasoulas, Ludovic Dos Santos, Kevin Scaman, and Aladin Virmaux. Coloring graph neural networks for node disambiguation. In *Proceedings of the Twenty-Ninth International Conference on International Joint Conferences on Artificial Intelligence*, pages 2126–2132, 2021.

[19] Holger Dell, Martin Grohe, and Gaurav Rattan. Lovász meets weisfeiler and leman. In *45th International Colloquium on Automata, Languages, and Programming (ICALP 2018)*. Schloss Dagstuhl-Leibniz-Zentrum fuer Informatik, 2018.

[20] Josep Díaz, Maria Serna, and Dimitrios M Thilikos. Counting h-colorings of partial k-trees. *Theoretical Computer Science*, 281(1-2):291–309, 2002.

[21] Reinhard Diestel. *Graph Theory*. Springer, 2017.

[22] Paul D Dobson and Andrew J Doig. Distinguishing enzyme structures from non-enzymes without alignments. *Journal of molecular biology*, 330(4):771–783, 2003.

[23] Zdeněk Dvořák. On recognizing graphs by numbers of homomorphisms. *Journal of Graph Theory*, 64(4):330–342, 2010.

[24] Jiarui Feng, Lecheng Kong, Hao Liu, Dacheng Tao, Fuhai Li, Muhan Zhang, and Yixin Chen. Extending the design space of graph neural networks by rethinking folklore weisfeiler-lehman. *Advances in Neural Information Processing Systems*, 36, 2024.

[25] Fabrizio Frasca, Beatrice Bevilacqua, Michael Bronstein, and Haggai Maron. Understanding and extending subgraph gnns by rethinking their symmetries. *Advances in Neural Information Processing Systems*, 35:31376–31390, 2022.

[26] Thomas Gärtner, Peter Flach, and Stefan Wrobel. On graph kernels: Hardness results and efficient alternatives. In *Learning Theory and Kernel Machines: 16th Annual Conference on Learning Theory and 7th Kernel Workshop, COLT/Kernel 2003, Washington, DC, USA, August 24-27, 2003. Proceedings*, pages 129–143. Springer, 2003.

[27] Floris Geerts. The expressive power of kth-order invariant graph networks. *arXiv preprint arXiv:2007.12035*, 2020.

[28] CD Godsil and BD McKay. A new graph product and its spectrum. *Bulletin of the Australian Mathematical Society*, 18(1):21–28, 1978.

[29] Martin Grohe. The complexity of homomorphism and constraint satisfaction problems seen from the other side. *Journal of the ACM (JACM)*, 54(1):1–24, 2007.

[30] Martin Grohe, Kristian Kersting, Martin Mladenov, and Pascal Schweitzer. Color refinement and its applications. *Van den Broeck, G.; Kersting, K.; Natarajan, S*, 30, 2017.

[31] Martin Grohe, Moritz Lichter, Daniel Neuen, and Pascal Schweitzer. Compressing cfi graphs and lower bounds for the weisfeiler-leman refinements. In *2023 IEEE 64th Annual Symposium on Foundations of Computer Science (FOCS)*, pages 798–809. IEEE, 2023.

[32] William L Hamilton. *Graph representation learning*. Morgan & Claypool Publishers, 2020.

[33] Pavol Hell and Jaroslav Nesetril. *Graphs and homomorphisms*, volume 28. OUP Oxford, 2004.

[34] Yinan Huang, Xingang Peng, Jianzhu Ma, and Muhan Zhang. Boosting the cycle counting power of graph neural networks with i$^2$-GNNs. In *The Eleventh International Conference on Learning Representations*, 2023. URL `https://openreview.net/forum?id=kDSmxOspsXQ`.

[35] Emily Jin, Michael Bronstein, Ismail Ilkan Ceylan, and Matthias Lanzinger. Homomorphism counts for graph neural networks: All about that basis. *arXiv preprint arXiv:2402.08595*, 2024.

[36] Nicolas Keriven and Gabriel Peyré. Universal invariant and equivariant graph neural networks. *Advances in Neural Information Processing Systems*, 32, 2019.

[37] Thomas N. Kipf and Max Welling. Semi-supervised classification with graph convolutional networks. In *International Conference on Learning Representations*, 2017. URL `https://openreview.net/forum?id=SJU4ayYgl`.

[38] Eric D Kolaczyk and Gábor Csárdi. *Statistical analysis of network data with R*, volume 65. Springer, 2014.

[39] Rui Li, Xin Yuan, Mohsen Radfar, Peter Marendy, Wei Ni, Terrence J O'Brien, and Pablo M Casillas-Espinosa. Graph signal processing, graph neural network and graph learning on biological data: a systematic review. *IEEE Reviews in Biomedical Engineering*, 16:109–135, 2021.

[40] Shouheng Li, Dongwoo Kim, and Qing Wang. Generalization of graph neural networks through the lens of homomorphism. *arXiv preprint arXiv:2403.06079*, 2024.

[41] Don R Lick and Arthur T White. k-degenerate graphs. *Canadian Journal of Mathematics*, 22 (5):1082–1096, 1970.

[42] Meng Liu, Hongyang Gao, and Shuiwang Ji. Towards deeper graph neural networks. In *Proceedings of the 26th ACM SIGKDD international conference on knowledge discovery & data mining*, pages 338–348, 2020.

[43] Andreas Loukas. What graph neural networks cannot learn: depth vs width. In *International Conference on Learning Representations*, 2020.

[44] László Lovász. Operations with structures. *Acta Mathematica Hungarica*, 18(3-4):321–328, 1967.

[45] László Lovász. *Large networks and graph limits*, volume 60. American Mathematical Soc., 2012.

[46] Yao Ma and Jiliang Tang. *Deep learning on graphs*. Cambridge University Press, 2021.

[47] Takanori Maehara and Hoang NT. A simple proof of the universality of invariant/equivariant graph neural networks. *arXiv preprint arXiv:1910.03802*, 2019.

[48] Takanori Maehara and Hoang NT. Learning on random balls is sufficient for estimating (some) graph parameters. *Advances in Neural Information Processing Systems*, 34:1126–1141, 2021.

[49] Shmoolik Mangan and Uri Alon. Structure and function of the feed-forward loop network motif. *Proceedings of the National Academy of Sciences*, 100(21):11980–11985, 2003.

[50] Haggai Maron, Heli Ben-Hamu, Nadav Shamir, and Yaron Lipman. Invariant and equivariant graph networks. In *International Conference on Learning Representations*, 2018.

[51] Haggai Maron, Heli Ben-Hamu, Hadar Serviansky, and Yaron Lipman. Provably powerful graph networks. *Advances in neural information processing systems*, 32, 2019.

[52] Ron Milo, Shai Shen-Orr, Shalev Itzkovitz, Nadav Kashtan, Dmitri Chklovskii, and Uri Alon. Network motifs: simple building blocks of complex networks. *Science*, 298(5594):824–827, 2002.

[53] Christopher Morris, Martin Ritzert, Matthias Fey, William L Hamilton, Jan Eric Lenssen, Gaurav Rattan, and Martin Grohe. Weisfeiler and leman go neural: Higher-order graph neural networks. In *Proceedings of the AAAI conference on artificial intelligence*, volume 33, pages 4602–4609, 2019.

[54] Ryan Murphy, Balasubramaniam Srinivasan, Vinayak Rao, and Bruno Ribeiro. Relational pooling for graph representations. In *International Conference on Machine Learning*, pages 4663–4673. PMLR, 2019.

[55] Daniel Neuen. Homomorphism-distinguishing closedness for graphs of bounded tree-width. *arXiv preprint arXiv:2304.07011*, 2023.

[56] Hoang NT and Takanori Maehara. Graph homomorphism convolution. In *International Conference on Machine Learning (ICML), Proceedings of Machine Learning Research. PMLR*, 2020.

[57] Raffaele Paolino, Sohir Maskey, Pascal Welke, and Gitta Kutyniok. Weisfeiler and leman go loopy: A new hierarchy for graph representational learning. *arXiv preprint arXiv:2403.13749*, 2024.

[58] Daniel Paul-Pena and C Seshadhri. A dichotomy theorem for linear time homomorphism orbit counting in bounded degeneracy graphs. *arXiv preprint arXiv:2211.08605*, 2022.

[59] Omri Puny, Derek Lim, Bobak T. Kiani, Haggai Maron, and Yaron Lipman. Equivariant polynomials for graph neural networks, 2023.

[60] Jiezhong Qiu, Jian Tang, Hao Ma, Yuxiao Dong, Kuansan Wang, and Jie Tang. Deepinf: Social influence prediction with deep learning. In *Proceedings of the 24th ACM SIGKDD international conference on knowledge discovery & data mining*, pages 2110–2119, 2018.

[61] David E Roberson. Oddomorphisms and homomorphism indistinguishability over graphs of bounded degree. *arXiv preprint arXiv:2206.10321*, 2022.

[62] T Konstantin Rusch, Michael M Bronstein, and Siddhartha Mishra. A survey on oversmoothing in graph neural networks. *arXiv preprint arXiv:2303.10993*, 2023.

[63] Ryoma Sato. A survey on the expressive power of graph neural networks. *arXiv preprint arXiv:2003.04078*, 2020.

[64] Ryoma Sato, Makoto Yamada, and Hisashi Kashima. Random features strengthen graph neural networks. In *Proceedings of the 2021 SIAM international conference on data mining (SDM)*, pages 333–341. SIAM, 2021.

[65] Franco Scarselli, Sweah Liang Yong, Marco Gori, Markus Hagenbuchner, Ah Chung Tsoi, and Marco Maggini. Graph neural networks for ranking web pages. In *The 2005 IEEE/WIC/ACM International Conference on Web Intelligence (WI'05)*, pages 666–672. IEEE, 2005.

[66] Ida Schomburg, Antje Chang, Christian Ebeling, Marion Gremse, Christian Heldt, Gregor Huhn, and Dietmar Schomburg. Brenda, the enzyme database: updates and major new developments. *Nucleic acids research*, 32(suppl_1):D431–D433, 2004.

[67] Behrooz Tahmasebi, Derek Lim, and Stefanie Jegelka. The power of recursion in graph neural networks for counting substructures. In *International Conference on Artificial Intelligence and Statistics*, pages 11023–11042. PMLR, 2023.

[68] Remco Van Der Hofstad. *Random graphs and complex networks*. Cambridge university press, 2024.

[69] S Vichy N Vishwanathan, Nicol N Schraudolph, Risi Kondor, and Karsten M Borgwardt. Graph kernels. *Journal of Machine Learning Research*, 11:1201–1242, 2010.

[70] Hinrikus Wolf, Luca Oeljeklaus, Pascal Kühner, and Martin Grohe. Structural node embeddings with homomorphism counts. *arXiv preprint arXiv:2308.15283*, 2023.

[71] Shiwen Wu, Fei Sun, Wentao Zhang, Xu Xie, and Bin Cui. Graph neural networks in recommender systems: a survey. *ACM Computing Surveys*, 55(5):1–37, 2022.

[72] Keyulu Xu, Weihua Hu, Jure Leskovec, and Stefanie Jegelka. How powerful are graph neural networks? In *International Conference on Learning Representations*, 2018.

[73] Gilad Yehudai, Ethan Fetaya, Eli Meirom, Gal Chechik, and Haggai Maron. From local structures to size generalization in graph neural networks. In *International Conference on Machine Learning*, pages 11975–11986. PMLR, 2021.

[74] Hanqing Zeng, Muhan Zhang, Yinglong Xia, Ajitesh Srivastava, Andrey Malevich, Rajgopal Kannan, Viktor Prasanna, Long Jin, and Ren Chen. Decoupling the depth and scope of graph neural networks. *Advances in Neural Information Processing Systems*, 34:19665–19679, 2021.

[75] Bingxu Zhang, Changjun Fan, Shixuan Liu, Kuihua Huang, Xiang Zhao, Jincai Huang, and Zhong Liu. The expressive power of graph neural networks: A survey. *arXiv preprint arXiv:2308.08235*, 2023.

[76] Bohang Zhang, Shengjie Luo, Liwei Wang, and Di He. Rethinking the expressive power of gnns via graph biconnectivity. In *The Eleventh International Conference on Learning Representations*, 2022.

[77] Bohang Zhang, Guhao Feng, Yiheng Du, Di He, and Liwei Wang. A complete expressiveness hierarchy for subgraph gnns via subgraph weisfeiler-lehman tests. *arXiv preprint arXiv:2302.07090*, 2023.

[78] Bohang Zhang, Jingchu Gai, Yiheng Du, Qiwei Ye, Di He, and Liwei Wang. Beyond weisfeiler-lehman: A quantitative framework for GNN expressiveness. In *The Twelfth International Conference on Learning Representations*, 2024. URL https://openreview.net/forum?id=HSKaGOi7Ar.

[79] Muhan Zhang and Pan Li. Nested graph neural networks. *Advances in Neural Information Processing Systems*, 34:15734–15747, 2021.

[80] Lingxiao Zhao, Wei Jin, Leman Akoglu, and Neil Shah. From stars to subgraphs: Uplifting any gnn with local structure awareness. In *International Conference on Learning Representations*, 2021.

---

**Algorithm 1** Algorithm for tree pattern $P$.

---

1: **procedure** RECURSION($P^\bullet$, $p$)
2:     $\mathrm{dp}_p[u] \leftarrow 0$ for all $u \in V(G)$
3:     **for** $q \in \mathrm{children}(p)$ **do**
4:         $\mathrm{dp}_q \leftarrow$ RECURSION($P^\bullet$, $q$)
5:         $\mathrm{dp}_p[u] \leftarrow \mathrm{dp}_p[u] + \mu_p(x_u) \sum_{v \in N(u)} \mathrm{dp}_q[v]$ for all $u \in V(G)$
6:     **end for**
7:     **return** $\mathrm{dp}_p$
8: **end procedure**

---

# A   Algorithms

## A.1   Algorithm for Bounded Treewidth Pattern

If the pattern graphs have the bounded treewidth, we can compute the generalised homomorphism numbers in a polynomial time. Since describing the general case requires some preparation about tree decomposition, we here present the algorithm for tree patterns in Algorithm 1. We emphasise that this algorithm is just for an illustrative purpose because tree patterns are generated from $\bullet - \circ$ so the standard message passing GNNs can capture these patterns.

As a preprocessing, we make the pattern $P^\bullet$ directed toward the root. Let $P^{\bullet,p}$ be the subtree of $P^\bullet$ rooted at $p$. The procedure RECURSION($P^\bullet$, $p$) computes the array $[\mathrm{hom}((P^{\bullet,p}, \mu), (G^u, x)) : u \in V(G)]$. By the chain rule (Lemma 5.3), we have the following recursive formula.

$$\mathrm{hom}((P^{\bullet,p}, \mu), (G^u, x)) = \mu_p(x_u) \sum_{q \in \mathrm{children}(p)} \sum_{v \in N(u)} \mathrm{hom}((P^{\bullet,q}, \mu), (G^v, x)). \qquad (10)$$

For each node $p$, RECURSION($P^\bullet$, $p$) is invoked exactly once. Thus, the complexity of the procedure is $O(|V(P^\bullet)||E(G)|)$, which is linear in $G$.

The generalisation to the bounded tree-width case is straight-forward — we just run a similar dynamic programming algorithm where the states are bags. As we need to maintain the mapping from $V(G)$ to the states, the complexity becomes $V(G)^{\mathrm{tw}(G)+1}$. See [20].

## A.2   Algorithm for Bounded DAG-Treewidth Pattern and Bounded Degeneracy Graph

If the pattern $P^\bullet$ has the bounded DAG-treewidth and $G$ has the bounded degeneracy, we can compute the generalised homomorphism number in a polynomial time. Since describing the general case requires lots of preparation about DAG-tree decomposition, we here present the linear-time algorithm for the simplest case that the pattern is the quadrange (aka. four cycle) $C_4$. This result is non-trivial because there will be $\Omega(n^2)$ quadrangles in a graph of bounded degeneracy (imagine the complete bipartite graph $K_{2,n}$, which has two left nodes and $n$ right nodes, has the degeneracy of two but has $\Theta(n^2)$ quadrangles); hence, any naive enumeration algorithm requires $\Omega(n^2)$ time.

We can observe that all quadrangles that have $u$ and $v$ as the opposite nodes is represented as a "compressed" format, $(u, v, \{w_1, \ldots, w_k\})$, meaning that there are $\binom{k}{2}$ quadrangles by choosing any two $w_i$ and $w_j$ in addition to $u$ and $v$. For example, in the above-mentioned $K_{2,n}$ case, we have only three tuples $(u, u, \{w_1, \ldots, w_n\})$, $(u, v, \{w_1, \ldots, w_n\})$, and $(v, v, \{w_1, \ldots, w_n\})$ to represent all quadrangles in the graph. Chiba and Nishizeki [14] observed that if the graph has bounded degeneracy, we obtain a linear-size compressed representation for all quadrangles in the graph. Their algorithm is presented in Algorithm 2.

Suppose we have a compressed representation of quadrangles $(u, v, \{w_1, \ldots, w_j\})$. Then, we can compute their contributions to node $u$ as

$$\sum_{i,j} \mu_1(x_u)\mu_2(x_{w_i})\mu_3(x_v)\mu_4(x_{w_j}) = \mu_1(x_u)\left(\sum_i \mu_2(x_{w_i})\right)\mu_3(x_v)\left(\sum_j \mu_4(x_{w_j})\right), \qquad (11)$$

---

**Algorithm 2** Chiba–Nishizeki algorithm for enumerating all homomorphic images of a quadrangle in a compressed format.

---

Initialise $\text{set}[u] = \varnothing$ for all $u \in V(G)$
**for** $u \in V(G)$ in the decreasing order of the degree **do**
    **for** $w \in N(u_i)$ **do**
        **for** $v \in N(w)$ **do**
            Insert $w$ to the $\text{set}[v]$
        **end for**
    **end for**
    **for** $v$ with $\text{set}[v] \neq \varnothing$ **do**
        Report $(u, v, \text{set}[v])$
        $\text{set}[v] \leftarrow \varnothing$
    **end for**
    Remove $u$ from $G$
**end for**

---

---

**Algorithm 3** Algorithm for evaluating $\text{hom}((C_4, \mu), (G^u, x))$ for all $u$

---

**for** $(u, v, \{w_1, \ldots, w_k\})$ produced by Algorithm 2 in Appendix **do**
    Compute $W_p := \sum_i \mu_p(x_{w_i})$ for $p = 2, 3, 4$.
    $z_u \leftarrow z_u + \mu_1(u)W_2\mu_3(x_v)W_4$
    $z_v \leftarrow z_v + \mu_1(v)W_2\mu_3(x_u)W_4$
    $z_w \leftarrow z_w + \mu_1(x_w)\mu_2(x_u)W_3\mu_4(x_v)$
**end for**
Report $z_u$ as $\text{hom}((C_4, \mu), (G^u, x))$

---

which is evaluated in $O(k)$ time. The same procedure is applied to the contributions to node $v$. We can also compute the contributions to $w_i$ by

$$\sum_j \mu_4(x_u)\mu_1(x_{w_i})\mu_2(x_v)\mu_3(x_{w_j}) = \mu_4(x_u)\mu_1(x_{w_i})\mu_2(x_v)\left(\sum_j \mu_2(x_{w_j})\right). \tag{12}$$

Here, as the $x_{w_j}$ factors are common in all $w_i$, we can evaluate them for all $i$ in $O(k)$ time in total. This procedure is summarised in Algorithm 3.

The general case (bounded DAG-treewidth and bounded degeneracy) is a far generalisation of the above idea [10, 58]. Let $\vec{P}$ be an DAG orientation of $P$. Then, a DAG tree decomposition of $\vec{P}$ is a tree of bags such that (1) each bag $B$ is a subset of source nodes (nodes without incoming edges), and (2) the union of bags covers all source nodes, and (3) if $B$ lies on the unique path between $B_1$ and $B_2$, then $\text{reachable}(B_1) \cap \text{reachable}(B_2) \subseteq \text{reachable}(B)$. The maximum size of the bag is called the DAG-treewidth. The dynamic programming algorithm on the DAG tree decomposition is similar to that on the tree decomposition case but enumerates all compressed representations of homomorphisms instead of the homomorphisms; see [10]. For example, $C_4 = \{1, 2, 3, 4\}$ has a DAG tree decomposition with two bags $\{1\}$ and $\{3\}$, and the dynamic programming with respect to this DAG tree decomposition produces Chiba–Nishizeki's compressed representation of all quadrangles. This dynamic programming is easily converted to compute the generalised homomorphism numbers.

# B  Related Work and Comparison with Our Model

There are multiple GNN models that attain the intermediate complexity between the universal GNN and the MPGNN. Here, we review some of these models and describe their relationship with our model.

## B.1  k-GNNs

The $k$-GNNs [53] or PPGN [51] assign values to $k$-tuples of nodes (instead of the nodes itself as in MPGNN), and the $k$-IGNs [50] use equivariant linear layers defined by $k$-th order tensors. They

have the same expressive power as the $k$-dimensional WL test [72, 27], which is equivalent to the $\mathcal{T}_k$-homomorphism indistinguishability where $\mathcal{T}_k$ is the graphs of treewidth at most $k$ [23]. A recent variant [24] reduced the space complexity to $O(n^2)$ while keeping the expressive hierarchy to the graph isomorphism problem; we used this model ($N^2$-GNN) in our Experiment. These models are often used in molecular biology applications as the input graphs are small.

The homomorphism characterisation mentioned above proves that $\mathcal{P}^\bullet$-DHN model is not more expressive than $k$-GNN and $k$-IGN models as in Corollary 5.11 and Corollary 5.12. On the other hand, $k$-GNN and $k$-IGN might require $\Omega(n^k)$ time as they have to aggregate information over $k$-tuples globally. Hence, they are not suited for large graphs as there will be millions or billions of nodes.

## B.2 Subgraph GNN

Subgraph GNNs [79, 74, 80, 25, 67] are designed to capture local structures of each node. A single layer of a subgraph GNN takes a subgraph for each node and applies a base GNN. The expressive power of the subgraph GNNs varies on the subgraph selection policy and the base GNN model. The common subgraph selection policies include $k$-hop egographs, node/edge marking, and node/edge deletion, and the common base GNN is the MPGNN. In this case, Frasca et al. [25] proved that its expressive power is bounded by 3-WL test. Zhang et al. [77] analysed the expressive power of the subgraph GNNs by introducing *subgraph WL-test*, which basically runs the WL test on each selected subgraphs. Huang et al. [34] analysed the expressive power of the subgraph GNNs and showed that if we use MPGNN as the base encoder, it cannot count cycles of length more than four. Huang et al. [34] then developed a variant of node-marking GNN that can count at least 6 cycles while maintaining a linear time complexity on graphs of bounded degree; we used this model in our Experiment ($I^2$-GNN). Tahmasebi et al. [67] showed a recursive subgraph selection policy has higher expressive power learning all local functions.

In general, subgraph GNNs do not fit the homomorphism framework. However, we can still analyse their properties using the homomorphism framework. Let us consider the subgraph GNN model that uses $k$-hop egograph selection and universal GNN; this model is more expressive than any subgraph GNN model that uses $k$-hop egograph selection policy. Let $\mathcal{B}_k^\bullet$ be the graphs of the radius from $\bullet$ at most $k^6$. Then, as homomorphism numbers from $\mathcal{B}_k^\bullet$ to $G^u$ characterise $k$-hop neighbours of $u$, the above model has the same expressive power as the $\mathcal{B}_k^\bullet$-DHN model. This characterisation proves Corollary 5.13, which was already known in [77, Theorem 7.1].

Subgraph GNNs perform local computation; hence, they run in $O(n)$ time on bounded degree graphs. However, they are still not suitable for large sparse graphs in the real world as these graphs contain a few nodes with very large degrees (*power-law property*) and a small diameter (*small-world property*) [3]. In such graphs, building $k$-hop egographs for all nodes may take $\Omega(n^2)$ time.

## B.3 GNN with explicit pattern detection

Using the numbers of subgraphs (incl. homomorphisms) as features is a traditional approach in network science and graph data mining [15, 69, 52, 26]. Recently, several researchers tried to integrate this technology in GNNs [47, 56, 4, 9, 78].

As we mentioned in Section 1, our work is strongly motivated by the model of NT and Maehara [56] and the theoretical analysis of [4]. The resulting model (or layer) provides a building block of GNNs that perform local aggregation.

Using graph neural network Applying the homomorphism theory (or subgraph enumeration) in GNN is a relatively new approach. These are classified as follows.

**Models based on graph homomorphisms**    To the best of our knowledge, NT and Maehara [56] is the only study that explicitly uses graph homomorphisms as a building block of a machine learning model in a GNN context. Beaujean et al. [5] proposed to sample homomorphisms to estimate homomorphism numbers to accelerate the computation. Related studies include the GNN with equivariant polynomials [36, 59] since the homomorphism numbers define equivariant polynomials [47, 59].

Recently, Paolino et al. [57] proposed a GNN architecture that uses cycles for aggregation. They proved that their model can count cactus graphs of bounded cycle lengths. Their model is a special case of our DHN using $C_{\leqslant k} := \{C_1, \ldots, C_k\}$ as the patterns, and the set of cactus graphs of bounded cycle lengths is exactly the set of graphs generated from $C_{\leqslant k}$ using the rooted product. In this sense, our method can be seen as a generalisation of their methods for arbitrary patterns. One minor but crucial difference is that they didn't use the feature transformation. This means that their model cannot distinguish a homogeneous cycle (adjacent nodes have similar features) and heterogeneous cycle (adjacent nodes have dissimilar features). See also Remark 3.2 about the importance of feature transformation in theory.

**Injecting homomorphisms numbers and/or subgraph counts as features** Barceló et al. [4] proposed injecting homomorphism numbers into the node features and Bouritsas et al. [9] proposed injecting subgraph counting into the node features. As these numbers are connected by the Mobius transformation [16], their expressive powers are not so different if we consider multiple patterns. Jin et al. [35] studied the difference and identified the effective set of patterns to be injected. We believe that their findings are useful for selecting the set of patterns $\mathcal{P}^{\bullet}$ in our DHN model.

**Analysing expressive power by homomorphisms** Traditionally, the expressive powers of GNNs have been studied using the Weisfeiler–Lehman test [72]. As the Weisfeiler–Lehman test has the homomorphism characterisation [19], it is natural to extend this discussion to more expressive GNNs. The above-mentioned studies of injecting counts Barceló et al. [4], Bouritsas et al. [9], Jin et al. [35] studied the expressive power of homomorphism numbers and/or subgraph counts-injected models via homomorphism numbers. Zhang et al. [78] identified the homomorphism characterisation of the expressive power of local WL-based GNNS. The pattern sets are characterised using ear decomposition. Li et al. [40] evaluated a generalised error of a GNN using homomorphism entropy. We believe that their analysis could be used to evaluate the generalisation error of our DHN model as well.

# C  Experiment Details

## C.1  Model Configurations

All DHN models in Table 1 have 20 hidden units MLP layers; these MLP blocks (3 layers) correspond to functions $\mu$ in Equation 3. Each homomorphism kernel is embedded in 10 dimensions. The DHN models are trained using the Adam optimizer with an initial learning rate of 0.001. We do not use any learning rate scheduling or advanced regularization techniques, as the expressivity benchmark datasets can be learned with default hyperparameters.

Homomorphism mappings are pre-computed for each input graph and loaded to DHN like the edge list is loaded to Pytorch Geometric's API. The homomorphism enumeration can be run in linear time and parallelizable for large graphs; hence, this pre-computation step is negligible compared to the training process. All our experiments can be run on a CPU machine due to the small model size (M3 chip with 24GB of memory shared with the operating system or CPU-type Google Colab instance). The reported results are obtained on a single GPU machine that houses an RTX4090 with 24GB of GPU memory.

## C.2  Datasets and Evaluations

Each experiment is run for a maximum of 1200 epochs. Early stopping on train set accuracy with patience of 10 epochs is used for ENZYME and PROTEINS. For CSL, EXP, and SR25, since they are expressivity benchmark datasets, the model is trained until train loss converged to zero.

Table 2 describes the three synthetic expressivity benchmark datasets commonly utilized in Graph Neural Network (GNN) research to assess and benchmark the expressive power of various GNN architectures. The CSL (Circular Skip Links) dataset consists of 150 regular graphs that cannot be distinguished by simple MPGNN. EXP graphs are crafted to be isomorphic under the 1-WL test, meaning that traditional GNNs limited by the WL test's discriminative power may fail to distinguish them. The EXP dataset is a benchmark for whether GNNs can surpass the WL test limitations by capturing higher-order structural information, distinguishing between non-isomorphic but 1-WL-indistinguishable graphs. The SR25 (also named Paulus Graphs) dataset consists of strongly regular

Table 2: Expressivity Benchmark Datasets

| Dataset | $|\mathcal{G}|$ | $|V(G)|$ | $|E(G)|$ |
|---------|------|----------|----------|
| CSL     | 150  | 41.0     | 164.0    |
| EXP     | 1200 | 44.4     | 110.2    |
| SR25    | 15   | 25.0     | 300.0    |

Table 3: Real-world Graph Classification Datasets

| Dataset  | $|\mathcal{G}|$ | $\mathbb{E}|V(G)|$ | $\mathbb{E}|E(G)|$ | #node features | #classes |
|----------|------|--------------------|--------------------|----------------|----------|
| PROTEINS | 1113 | 39.06              | 72.82              | 4              | 2        |
| ENZYMES  | 600  | 41.0               | 164.0              | 21             | 6        |

and co-spectral graphs, which require high expressivity GNNs to distinguish. Essentially, a model that performs perfectly on the train set of this dataset would perform well on the test set because both sets have the same isomorphism classes. Except for SR25, which needed 800 epochs to converge, EXP and CSL training converged in less than 20 epochs in our experiments.

Table 3 describes the real-world datasets commonly utilized in Graph Neural Network (GNN) research to assess the practicality of a graph neural network. These datasets come from the TUDatasets collection, and due to their small size, it is conventional to report their 10-fold cross-validation results. The PROTEINS dataset consists of 1113 graphs, where the nodes and edges of each graph contain information about the secondary structure of the protein. The ENZYME dataset contains 600 graphs, each corresponding to a protein enzyme. The nodes signify amino acids, and the edges represent chemical interactions or spatial proximities between these amino acids. The dataset is divided into six classes, each corresponding to one of the top-level enzyme categories defined by the Enzyme Commission (EC) numbers: oxidoreductases, transferases, hydrolases, lyases, isomerases, and ligases. Nodes are annotated with attributes capturing physicochemical properties relevant to protein function.

# D   Proofs

## D.1   Proof of Theorem 3.1: Generalised Homomorphism Determines Isomorphism

We use the following generic theorem.

**Theorem D.1** (Theorem 3.6 in Lovasz [44]). *Any finite relational structure is uniquely identified by the homomorphism numbers.*

*Proof.* The "if" direction is clear. Thus, we prove the "only-if" direction.

We first enumerate the relevant feature vectors $\{x_{1u_2} : u_1 \in V(G_1)\} \cup \{x_{2u_2} : u_2 \in V(G_2)\}$ and associate unique labels to them. We denote by $l(x)$ for the label associated with $x$. Then, the input graphs are the instances of finite relational structure, where the relations are the root relation ($u$ is the root), adjacency relation ($u$ and $v$ have edges), and the feature value relation ($u$ has the feature value $x$).

The isomorphism $(G_1^\bullet, x_1) \simeq (G_2^\bullet, x_2)$ of rooted featured graphs coincides with the isomorphism of the relational structure introduced the above. By the Lovasz theorem, if the input graphs are not isomorphism, there exists a relational structure $(F^\bullet, l)$ such that the homomorphism numbers from this structure distinguishes the input graphs. Because the homomorphism number of this relational structure can be computed by the generalised homomorphism number, by setting (a smoothed version of) $\mu_p(x) = 1[l(x) = l_p]$. Therefore, we obtain the result. $\square$

## D.2 Remark 5.8: DHN generalises the most expressive subgraph GNNs

Suppose $h_u = f(h_v : v \in H^u)$. Let $P^\bullet$ be the graph isomorphic to $H^u$. Then, we have

$$h_u = \frac{1}{|\mathrm{Aut}(G^u)|} \sum_{\pi \in \mathrm{Aut}(G^u)} f\left(h_v : v \in \pi(H^u)\right) \tag{13}$$

$$\propto \sum_{\pi \in \mathrm{Hom}^{(\mathrm{inj})}(P^\bullet, G^u)} f\left(h_{\pi(p)} : p \in P^\bullet\right) \tag{14}$$

$$= \rho\left(\sum_{\pi \in \mathrm{Hom}^{(\mathrm{inj})}(P^\bullet, G^u)} \prod_{p \in P^\bullet} \mu_p(h_{\pi(p)})\right) \tag{15}$$

$$= \rho\left(\mathrm{hom}^{(\mathrm{inj})}((P^\bullet, \mu), (G^u, h))\right), \tag{16}$$

because the first equality follows from the equivariance of the layer, where $\mathrm{Aut}(G^u)$ is the set of automorphisms (isomorphisms to itself) of $G^u$, the second proportionality follows because each automorphism induces an injective homomorphism, and the third equality follows by taking $\mu_p(x) := \exp([0, \ldots, 0, x, 0, \ldots, 0])$ and $\rho(z_1, \ldots, z_{|P^\bullet|}) := f(\log z_p : p \in P^\bullet)$.

It should be noted that this does not cover strategies like node marking in subgraph GNNs as such strategies (tentatively) break isomorphisms to improve their expressive power. To cover such strategies, we might need a higher-order theory of DHN.

## D.3 Proof of Theorem 5.2: Expressive power of DHN

To prove the theorem, we introduce a variant of the WL test as follows. The $\mathcal{P}^\bullet$-*Weisfeiler Lehman test* performs the following colour-refinement procedure. In the 0-th step, we assign the node features as the colour. In the $(k+1)$-th step, for each $u \in V(G)$, it enumerates all patterns $P^\bullet$ and all rooted homomorphisms $\pi \in \mathrm{Hom}(P^\bullet, G^u)$, and associates the colours based on the colours in the $k$-th step. Formally, it is given as follows.

$$c_u^{(0)} = x_u, \quad c_u^{(k+1)} = \left(\left\{\!\!\left\{\left(c_{\pi(p)}^{(k)} : p \in V(P^\bullet)\right) : \pi \in \mathrm{Hom}(P^\bullet, G^u)\right\}\!\!\right\} : P^\bullet \in \mathcal{P}^\bullet\right). \tag{17}$$

Then, it determines the non-isomorphism using the obtained colours, like the WL test.

**Example D.2.** $\{\bullet, \bullet - \circ\}$-WL test coincides with the standard WL test.

Now, we state our main theorem about the expressive power of the DHN model.

**Theorem D.3.** *Let $\mathcal{P}^\bullet$ be a set of rooted graphs. For two rooted graphs with features $(G_1^\bullet, x_1)$ and $(G_2^\bullet, x_2)$, the following are equivalent.*

1. *The $\mathcal{P}^\bullet$-WL does not distinguish $(G_1^\bullet, x_1)$ and $(G_2^\bullet, x_2)$.*

2. *For any $\mathcal{P}^\bullet$-DHN $h$, we have $h(G_1^\bullet, x_1) = h(G_2^\bullet, x_2)$.*

3. *$(G_1^\bullet, x_1)$ and $(G_2^\bullet, x_2)$ are $\overline{\mathrm{P}^\bullet}$-homomorphism indistinguishable.*

**(1 $\Rightarrow$ 2)**. This part is a generalisation of Theorem 3 in [72]. This is trivial from the definitions because the WL-colouring contains all information that is needed to compute the DHN.

**(2 $\Rightarrow$ 3)**. We show that, for all $F^\bullet \in \mathcal{F}^\bullet$ and $\mu$, there exists a $\mathcal{P}$-DHN $h$ such that $\mathrm{hom}((F, \mu), (G^u, x)) = h(u)$, which immediately proves this claim. We prove this claim by the induction about the construction of $F$.

**Base Case** The base case is that $F$ is a pattern graph, i.e., $F = P$ for some $P \in \mathcal{P}$. This case is trivial from the definition of the DHN model.

**Induction Case** Induction case is that $F$ is obtained by attaching smaller subpatterns $F_1, \ldots, F_N \in \mathcal{F}$ to some $P \in \mathcal{P}$. By the product rule and the chain rule, we can represent $\mathrm{hom}((F, \mu), G)$ by the sum and product of $\mathrm{hom}((F_i, \mu_i), G)$, which are represented by the DHN by the inductive hypothesis.

As the DHN is closed under sum and product (Lemma D.6), $\hom((F, \mu), G)$ is also represented by a DHN.

**($3 \Rightarrow 1$).** This claim is a generalisation of [19]. Here, we provide "direct" proof of this claim. We prove that $\mathcal{P}_\bullet$-WL colouring of step $k$ is identified from the values $\hom((F_j, \mu_j), G^u)$ for some $(F_i, \mu_i) \in \mathcal{F}$. In the proof, we extensively use the following lemma.

**Lemma D.4.** *The values of the multi-symmetric power sum polynomials,*

$$\sum_{i=1}^{n} \prod_{j=1}^{d} a_{ij}^{e_j}, \tag{18}$$

*uniquely determine the multiset of vectors*

$$\{\!\{(a_{i1}, \ldots, a_{id}) : i = 1, \ldots, n\}\!\}. \tag{19}$$

*Here, each $e_j$ is bounded by a constant that depends on $n$ and $d$.*

*Proof.* This is an extension of the famous "fundamental theorem of symmetric polynomials" and follows from the basic results of the invariant theory; see [11, Theorem 3]. $\square$

First of all, we can assume that we know all the values $\{x_p\}_{p \in V}$ relevant to the computation. This is because all the nodes that appeared in the computation are contained in a sufficiently large homomorphic image $F_\bullet \in \mathcal{P}$, and by putting $\mu_p(x) = \prod_j x_j^{e_{pj}}$, we have

$$\hom_\mu(F_\bullet, G_u) = \sum_{\pi \in \mathrm{Hom}(F_\bullet, G_u)} \prod_{p \in V(F_\bullet)} \prod_{j \in [d]} (x(\pi(p))_j)^{e_{pj}}. \tag{20}$$

Therefore, by Lemma D.4, we can uniquely reconstruct the set

$$X := \{x_{\pi(p)} : p \in V(P^\bullet)\}, \tag{21}$$

which tells all the possible values that appeared in the computation.

Now, we prove the claim by the induction about the depth $k$ of the pattern expansion.

**Base Case** The case $k = 0$ is trivial.

**Induction Case** We first identify the number of children with respect to $P$ by computing

$$\hom((P, \mu_{i_1, \ldots, i_{|V(P)|}}), G^u) \tag{22}$$

for $\mu_{i_1, \ldots, i_{|V(P)|}}$ that satisfies

$$\prod_{p \in V(P)} \mu_{i_1, \ldots, i_{|V(P)|}, p}(x_p) \neq 0$$

$$\iff x_{p_1} = x_{i_1}, \ldots, x_{p_{|V(P)}} = x_{i_{|V(P)|}}. \tag{23}$$

Such $\mu$ can be constructed by projecting vectors into $|V(P)|N$ dimensional space.

Next, we identify the colour of the substructures below $p \in V(P)$. By the induction hypothesis, there exist patterns with transformations $(F_1, \mu_1), \ldots, (F_M, \mu_M)$ such that their homomorphisms identify the colour of steps less than $k$. Using these patterns with transformations, we define a new set of patterns with transformations as follows.

- For each $p \in V(F)$, we attach $F_1$ $e_{p,1}$ times, $F_2$ $e_{p,2}$ times, and so on, by the rooted product.

By definition, this pattern is in $\mathcal{F}$. By the chain rule of homomorphism numbers, we have the following.

$$\hom((F, \mu), G^u) = \sum_{\pi \in \mathrm{Hom}(P, G^u)} \prod_{\substack{p \in V(P), \\ i=1, \ldots, M}} \hom((F_i, \mu_i), G^{\pi(p)})^{e_{p,i}} \tag{24}$$

By Lemma D.4, we can uniquely reconstruct the multiset of vectors

$$\{(\hom((F_1, \mu_1), G^{\pi(p_1)}), \ldots, \hom((F_M, \mu_M), G^{\pi(p_{|V(P)|})}))\}_{\pi \in \mathrm{Hom}(P, G)}. \tag{25}$$

Therefore, we can identify the colour of each $p$ in each child.

## D.4 Proof of Lemma 5.3: Chain Rule

*Proof.* Any rooted homomorphism $\pi \in \mathrm{Hom}(F^\bullet, G^\bullet)$ is identified as a concatenation of homomorphisms $\pi_0 \in \mathrm{Hom}(P^\bullet, G^\bullet)$ and $\pi_p \in \mathrm{Hom}(F_p, G^{\pi(p)})$ for each $p$. Hence,

$$\mathrm{hom}((F^\bullet, \mu), (G^\bullet, x)) \tag{26}$$

$$= \sum_{\pi \in \mathrm{Hom}(F^\bullet, G^\bullet)} \prod_{p \in V(F^\bullet)} \mu_p(x_{\pi(p)}) \tag{27}$$

$$= \sum_{\pi_0 \in \mathrm{Hom}(P^\bullet, G^\bullet)} \prod_{p \in V(P^\bullet)} \sum_{\pi_p \in \mathrm{Hom}(F_p^\bullet, G^{\pi(p)})} \prod_{q \in V(F_p^\bullet)} \mu_q(x_{\pi_p(q)}) \tag{28}$$

$$= \sum_{\pi_0 \in \mathrm{Hom}(P^\bullet, G^\bullet)} \prod_{p \in V(P^\bullet)} \mathrm{hom}((F_p^\bullet, \mu|_{V(F_p^\bullet)}), (G^{\pi(p)}, x)). \tag{29}$$

$\square$

## D.5 Proof of Theorem 5.16: Universality

This is a direct application of the Stone–Weierstrass theorem:

**Theorem D.5** (Stone–Weierstrass Theorem). *Let $\mathcal{M}$ be a compact Hausdorff space and $\mathcal{A}$ be a set of continuous functions. If $\mathcal{A}$ forms an algebra, contains the constant function and separates points, then $\mathcal{A}$ is dense in the space of uniformly continuous functions on $\mathcal{M}$.* $\square$

We first check that DHN forms an algebra.

**Lemma D.6.** *The $\mathcal{P}$-DHN model is closed under sum, product, and scalar multiplication.*

*Proof.* The claim is clear for scalar multiplication. To prove the claim for sum and product, we observe that the stacking of two models, i.e., $h(u) := [h_1(u), h_2(u)]$, is in the DHN as it is implemented by stacking every $\mu$ and $\rho$. The sum and product are obtained by modifying the last layer of the stacking model. $\square$

We then check the topological condition. Let $\mathcal{G}_d^\bullet$ be the set of all rooted graphs with features such that the maximum degree is at most $d$. We introduce the metric in this space as follows. Without loss of generality, we assume that $\max_{x_1, x_2 \in \mathcal{X}} \|x_1 - x_2\| \leqslant 1$ where $\|\cdot\|$ is the norm associated with $\mathcal{X}$. First, for each integer $r$, we define

$$d_r((G_1^\bullet, x_1), (G_2^\bullet, x_2)) := \begin{cases} 1, & G_1^\bullet \not\simeq G_2^\bullet, \\ \min_\pi (1/n_r) \sum_u \|x_{1u} - x_{2\pi(u)}\|, & \text{otherwise} \end{cases} \tag{30}$$

where $\pi$ runs over the isomorphism between $G_1^\bullet$ and $G_2^\bullet$, and $n_r$ is the maximum number of nodes of graphs of diameter $r$ and degree $d$. We then define

$$d((G_1^\bullet, x_1), (G_2^\bullet, x_2)) = \sum_r 2^{-r} d_r((G_1^\bullet, x_1)[N_r], (G_2^\bullet, x_2)[N_r]) \tag{31}$$

where $(G^\bullet, x)[N_r]$ is the graph with features whose graph part is the subgraph of $G^\bullet$ induced by the $r$-neighbourhood of the root and the feature part is the restriction on $N_r$. Note that this induces the standard Benjamini–Schramm topology if the graphs have no features [68]. We prove Theorem 5.16 with respect to this topology.

**Lemma D.7.** $\mathcal{G}_d^\bullet$ *is totally bounded Hausdorff.*

*Proof.* It is easy to see the space is Hausdorff because, as $d((G_1^\bullet, x_1), (G_2^\bullet, x_2)) = 0$ implies their $r$-neighbourhood are isomorphic for all $r$. By choosing $r$ sufficiently large as it covers whole $G_1^\bullet$ and $G_2^\bullet$, we obtain $G_1^\bullet \simeq G_2^\bullet$. By definition, there exists $\pi$ that satisfies $x_{1u} = x_{2\pi(u)}$ for all $u$. This indicates that $(G_1^\bullet, x_1) \simeq (G_2^\bullet, x_2)$.

Now we prove that the space is totally bounded by constructing an $O(\epsilon)$-net for any $\epsilon > 0$. We choose $r_0 \geqslant \log_2(1/\epsilon)$ and enumerate all rooted graphs of diameter at most $r_0$ and degree at most $d$. There are $O(2^{d^{r_0}})$ graphs and each of them have $O(d^{r_0})$ nodes. Then, for each rooted graph, we enumerate all the rooted graphs with features by assigning node features from the $\epsilon$-net of $\mathcal{X}$. Then, we obtain the $\epsilon$-net of $\mathcal{G}_d$. $\square$

**Corollary D.8.** *The covering number of $1$-Lipschitz functions on $\mathcal{G}_d^\bullet$ is $2^{(1/\epsilon)^{(1/\epsilon)^{O(1)}}}$.*

*Proof.* By counting the size of $O(\epsilon)$-net in the above proof, we see that the covering number is $(1/\epsilon)^{(1/\epsilon)^{O(1)}}$. Therefore, the covering number of 1-Lipschitz continuous functions is $2^{(1/\epsilon)^{(1/\epsilon)^{O(1)}}}$. $\square$

**Lemma D.9.** *Any $\mathcal{P}$-DHN $h$ is uniformly continuous on $\mathcal{G}_d^\bullet$.*

*Proof.* By definition, there exists a finite $r$ such that the $r$-neighbour of the root determines the value of $h$. We choose $\delta$ sufficiently small so that $d((G_1^\bullet, x_1), (G_2^\bullet, x_2)) < \delta$ indicates that the $r$-neighbour of $G_1^\bullet$ and $G_2^\bullet$ are topologically isomorphic. In this case, $h((G_1^\bullet, x_1))$ and $h((G_2^\bullet, x_2))$ only differ at the feature values. Therefore, for any $\epsilon > 0$, by choosing $\delta$ sufficiently small, we can make $|h((G_1^\bullet, x_1)) - h((G_2^\bullet, x_2))| < \epsilon$ for all $(G_1^\bullet, x_1)$ and $(G_2^\bullet, x_2)$ with $d((G_1^\bullet, x_1), (G_2^\bullet, x_2)) < \delta$. $\square$

*Remark* D.10. In this study, we only discuss the universality of the graphs of arbitrary sizes. We can easily prove the universality for graphs of a fixed size, which is more often discussed in previous studies (see [64, 75]), by the same proof strategy without any complicated topology discussion. However, we believe that such universality is not useful in practice as we usually apply GNNs to graphs of different number of nodes.

## D.6 Proof of Corollary 5.9: Comparison of Models

Let $\mathcal{P}_1^\bullet$ and $\mathcal{P}_2^\bullet$ be two set of patterns. Our goal is to understand the relationship between the equivalence relation of the $\overline{\mathcal{P}_1^\bullet}$-homomorphism indistinguishability and $\overline{\mathcal{P}_2^\bullet}$-homomorphism indistinguishability. Here, we provide a tool to identify their relationship.

Roberson [61] introduced a concept called homomorphism-distinguishing closed. A set of graphs $\mathcal{F}$ is *homomorphism-distinguishing closed* if a graph $H$ satisfies $\hom(H, G_1) = \hom(H, G_2)$ for all $G_1$ and $G_2$ with $\hom(F, G_1) = \hom(F, G_2)$ for all $F \in \mathcal{F}$ then $H \in \mathcal{F}$.

Here, we use the rooted graph variant as follows. A set of rooted graphs $\mathcal{F}^\bullet$ is *homomorphism-distinguishing closed* if a rooted graph $H^\bullet$ satisfies $\hom(H^\bullet, G_1^\bullet) = \hom(H^\bullet, G_2^\bullet)$ for all $G_1^\bullet$ and $G_2^\bullet$ with $\hom(F^\bullet, G_1^\bullet) = \hom(F^\bullet, G_2^\bullet)$ for all $F^\bullet \in \mathcal{F}^\bullet$ then $H^\bullet \in \mathcal{F}^\bullet$. By the definition, if $\mathcal{F}_1^\bullet \subsetneq \mathcal{F}_2^\bullet$ and they are homomorphism-distiguishing closed, then the homomorphism-distinguishability of $\mathcal{F}_2^\bullet$ leads a strictly finer equivalence relation than that of $\mathcal{F}_1^\bullet$. For a set of rooted graphs $\mathcal{F}^\bullet$, the *homomorphism-distinguishing closure* is the smallest homomorphism distinguishing set including $\mathcal{F}^\bullet$, which is well-defined. By definition, $\mathcal{F}^\bullet$ and $\mathrm{cl}(\mathcal{F}^\bullet)$ leads the same equivalence relation.

The following is the key lemma of connecting the homomorphism-distinguishing closedness on graphs with features and graphs without features.

**Theorem D.11.** *Let $\mathcal{F}^\bullet$ be a homomorphism-distinguishing closed set of rooted graphs. $H^\bullet \notin \mathcal{F}^\bullet$ if and only if there exists $(G_1^\bullet, x_1)$ and $(G_2^\bullet, x_2)$ such that $\hom((F^\bullet, \mu), (G_1^\bullet, x_1)) = \hom((F^\bullet, \mu), (G_2^\bullet, x_2))$ for all $F^\bullet \in \mathcal{F}^\bullet$ and $\mu$ but $\hom((H^\bullet, \nu), (G_1^\bullet, x_1)) \neq \hom((H^\bullet, \nu), (G_2^\bullet, x_2))$ for some $\nu$.*

The "only-if" direction is easy: $H^\bullet \notin \mathcal{F}^\bullet$ guarantees the existence of $G_1$ and $G_2$; thus, the claim holds for $(G_1, x)$ and $(G_2, x)$ where $x$ takes the same value at all nodes.

To prove the "if" direction, we use the following lemma.

**Lemma D.12.** *Let $(F^\bullet, \mu)$ be a rooted graph with transformations. Suppose $\hom((F^\bullet, \mu), (G_1^\bullet, x_1)) \neq \hom((F^\bullet, \mu), (G_2^\bullet, x_2))$ for some $(G_1^\bullet, x_1)$ and $(G_2^\bullet, x_2)$. Then, there exist $G_1''^\bullet$ and $G_2''^\bullet$ such that*

1. *$\hom(F^\bullet, G_1''^\bullet) \neq \hom(F^\bullet, G_2''^\bullet)$.*

2. *For any rooted graph $F'^\bullet$, $\hom(F'^\bullet, G_i''^\bullet)$ is the sum of $\hom((F'^\bullet, \mu'), (G_i^\bullet, x_i))$ for finitely many $\mu'$.*

*Proof.* To simplify the presentation, we prove the same claim for non-rooted graphs. We first construct node-weighted graphs $(K_i, w_i)$ for $i = 1, 2$ by defining $V(K_i) = \{(u_i, q) : u_i \in V(G_i), p \in V(F)\}$ and $E(K_i) = \{((u_i, p), (v_i, q)) : (u_i, v_i) \in E(G_i)\}$. We then put node weight

by $w((u_i, p)) = \gamma_p \mu_p(x_u)$ where $\{\gamma_p : p \in V(F)\}$ are algebraically independent numbers. Then, we have $\hom(F, (K_1, w_1)) \neq \hom(F, (K_2, w_2))$ since the $\prod_p \gamma_p$ term of $\hom(F, (K_i, w_i))$ is $\hom((F, \mu), (G_i, x_i))$.

We then construct unweighted graphs $H_i$ for $i = 1, 2$. We take a scale parameter $\alpha$ and define $K_i^\alpha$ by $V(K_i^\alpha) = \{(x_i, s) : x_i \in V(K_i), s \in \{1, \ldots, \lceil \alpha w(x_i) \rceil\}\}$ and $E(H_i^\alpha) = \{((x_i, s), (y_i, t)) : x_i \in V(K_i)\}$. Then, by construction, we see $\lim_{\alpha \to \infty} \hom(F, H_i^\alpha)/\alpha^{|V(F)|} = \hom(F, (K_i, w_i))$. This implies that for sufficiently large $\alpha$, we have $\hom(F, H_1^\alpha) \neq \hom(F, H_2^\alpha)$. This shows the first condition.

These $H_i^\alpha$ satisfy the second condition as follows:

$$\hom(F', H_i^\alpha) = \hom(F', (K_i, \lceil \alpha w_i \rceil)) \tag{32}$$

$$= \sum_{\pi \in \mathrm{Hom}(F', G_i)} \sum_{\iota : V(F') \to V(F)} \prod_{p' \in V(F')} \lceil \mu_{\iota(p')}(x_{i\pi(p')}) \rceil \tag{33}$$

$$= \sum_{\mu'} \hom((F', \mu'), (G_i, x_i)) \tag{34}$$

where $\mu'$ is defined by $\mu'_{p'} = \lceil \mu_{\iota(p')} \rceil$ for each $\iota : V(F') \to V(F)$.

We can easily modify the above proof to rooted graphs by only duplicating non-rooted nodes, although the notation will get messy. $\square$

*Proof of Theorem D.11 (if direction).* Suppose there exist $(G_1^\bullet, x_1)$ and $(G_2^\bullet, x_2)$ such that $\hom((F^\bullet, \mu_1), (G_1^\bullet, x_1)) = \hom((F^\bullet, \mu_1), (G_2^\bullet, x_2))$ for all $F^\bullet \in \mathcal{F}_1^\bullet$ and $\mu_1$, but $\hom((H^\bullet, \mu_2), (G_1, x_1)) \neq \hom((H^\bullet, \mu_2), (G_2^\bullet, x_2))$. We take $G_1'^\bullet$ and $G_2'^\bullet$ constructed in Lemma D.12. Then, by the first condition, we have $\hom(F_2^\bullet, G_1'^\bullet) \neq \hom(F_2^\bullet, G_2'^\bullet)$. Also, by the second condition, we have $\hom(F_1^\bullet, G_1'^\bullet) = \hom(F_1^\bullet, G_2'^\bullet)$ for all $F_1^\bullet \in \mathcal{F}_1^\bullet$. Therefore, as $\mathcal{F}^\bullet$ is homomorphism-distinguising closed, we have $H^\bullet \notin \mathcal{F}^\bullet$. $\square$

To compare the expressive powers of two DHN models with respect to patterns $\mathcal{P}_1^\bullet$ and $\mathcal{P}_2^\bullet$, we have to compare $\mathrm{cl}(\overline{\mathcal{P}_1^\bullet})$ and $\mathrm{cl}(\overline{\mathcal{P}_2^\bullet})$. However, it is not easy to characterise the homomorphism-distinguishing closure of a given set. Here, instead of characterising their closures, we try to find another homomorphism-separating set $\mathcal{H}^\bullet$ that "separates" $\overline{\mathcal{P}_1^\bullet}$ and $\overline{\mathcal{F}_2^\bullet}$ using the following lemma.

**Lemma D.13.** *Let $\mathcal{F}_1^\bullet$ and $\mathcal{F}_2^\bullet$ be sets of rooted graphs. If there is a homomorphism-distinguishing closed set $\mathcal{H}^\bullet$ such that $\mathcal{F}_1^\bullet \subseteq \mathcal{H}^\bullet$ and $\mathcal{F}_2^\bullet \nsubseteq \mathcal{H}^\bullet$, then $\mathrm{cl}(\mathcal{F}_1^\bullet) \nsubseteq \mathrm{cl}(\mathcal{F}_2^\bullet)$.*

*Proof.* This is clear from the closure property. $\square$

In the literature on homomoprhism distinguishability, we have several examples of homomorphism-distinguishing closed sets. Here, we use the following two examples of homomorphism-distinguishing closed sets.

- The set $\mathcal{T}_k$ of graphs of treewidth at most $k$ [61, 55].
- The set $\mathcal{CC}_k$ of graphs of maximum chordless cycle length at most $k$ [61].

As their homomorphism-distinguishabilities are proved for non-rooted graphs, we have to prove the corresponding result for rooted graphs. We use the following lemma, which guarantees that the rooted counterparts, $\mathcal{T}_k^\bullet$ and $\mathcal{CC}_k^\bullet$, are homomorphism-distinguishing closed.

**Lemma D.14.** *Let $\mathcal{F}$ be a homomorphism-distinguishing closed set. If $\mathcal{F}$ is closed under the rooted product (taking any node as root) and contains a single edge $\bullet - \circ$, then $\mathcal{F}^\bullet$ is homomorphism-distinguishing closed, where $\mathcal{F}^\bullet = \{F^u : F \in \mathcal{F}, u \in V(F)\}$ is the graphs obtained by picking each node as root.*

To prove this lemma, we use the following to connect the homomorphism number and the rooted homomorphism number.

**Lemma D.15.** *Let $\mathcal{F}^\bullet$ be a set of rooted graphs containing $\bullet - \circ$. Suppose $G$ is connected. Then, we can identify $\hom(F, G)$ for any $F \in \mathcal{F}$ by $\hom(H^\bullet, G^\bullet)$ for several $H^\bullet$.*

*Proof.* To identify $\hom(F, G)$, we construct $H^\bullet_{k,e} \in \mathcal{F}^\bullet$ as follows. We first construct a path of length $k$ by attaching edges. Then, for each node $i$, we attach $e_i$ edges (so it is a caterpillar graph). Finally, we attach $e_{k+1}$ copies of $F^\bullet$ to the tail (farthest point from the root) of the caterpiller. Then, we have

$$\hom(H^\bullet_{k,e}, G^\bullet) = \sum_{(u_1,\ldots,u_k):\text{walk from }\bullet} d_{u_1}^{e_1} \ldots d_{u_k}^{e_k} \hom(F^\bullet, G^{u_k})^{e_{k+1}}. \tag{35}$$

Hence, by Lemma D.4, we can obtain the following quantity:

$$\sum_{(u_1,\ldots,u_k):\text{walk from }\bullet} \frac{\hom(F^\bullet, G^{u_k})}{d_{u_1} \ldots d_{u_k}}. \tag{36}$$

Let $W = D^{-1}A$ be the random-walk matrix of $G$. Then, the above quantity is written as $e_\bullet^\top W^k[\hom(F^\bullet, G^u) : u \in V(G)]$, where $e_\bullet$ is the unit vector. Therefore, we can also obtain all the values $e_\bullet^\top((I+W)/2)^k[\hom(F^\bullet, G^u) : u \in V(G)]$. As $(I+W)/2$ defines an irreducible aperiodic random walk, $e_\bullet^\top((I+W)/2)^k$ converges to the stationary distribution $1/n$. Therefore, we can estimate $1'[\hom(F^\bullet, G^u) : u \in V(G)] = \hom(F, G)$ arbitrarily accurately. □

*Proof of Lemma D.14.* Let $H^\bullet$ be a rooted graph. Suppose there exists connected rooted graphs $G_1^\bullet$ and $G_2^\bullet$ such that $\hom(F^\bullet, G_1^\bullet) = \hom(F^\bullet, G_2^\bullet)$ for all $F^\bullet \in \mathcal{F}^\bullet$ but $\hom(H^\bullet, G_1^\bullet) \neq \hom(H^\bullet, G_2^\bullet)$. By Lemma D.15, the former condition yields $\hom(F, G_1) = \hom(F, G_2)$ for all $F \in \mathcal{F}^\bullet$ and the latter condition yields $\hom(H, G_1) \neq \hom(H, G_2)$. Therefore, by the homomorphism-distinguishing closedness of $\mathcal{F}$, we have $H \notin \mathcal{F}$. Therefore, $H^\bullet \notin \mathcal{F}^\bullet$. □

*Proof of Corollary 5.9.* We say that $\mathcal{H}^\bullet$ separates $\overline{\mathcal{P}_1^\bullet}$ from $\overline{\mathcal{F}_2^\bullet}$ if they satisfy the condition in Lemma D.13. In this case, by Theorem 5.2, we can conclude that $\mathcal{P}_1^\bullet$-DHN model is not more expressive than $\mathcal{P}_2^\bullet$-DHN model. Here, we can easily observe the following.

- MPGNN model is the $\{\bullet - \circ\}$-DHN model and $\mathcal{T}_1^\bullet$ separates $\overline{\{\bullet - \circ\}}$ from $\overline{\mathcal{P}^\bullet}$.

- 2-GNN model is the $\mathcal{T}_2^\bullet$-DHN model. Let $p$ be the longest chordless cycle length in $\mathcal{P}^\bullet$. Then, $\mathcal{CC}_{p+1}^\bullet$ separates $\overline{\mathcal{P}^\bullet}$ from $\mathcal{T}_2^\bullet$.

- $\mathrm{CC}_k^\bullet$ separates $\overline{\mathcal{C}_k^\bullet}$ from $\overline{\mathcal{C}_{k+1}^\bullet}$. $\mathcal{T}_k^\bullet$ separates $\overline{\mathcal{K}_k^\bullet}$ from $\overline{\mathcal{K}_{k+1}^\bullet}$. $\mathcal{T}_k^\bullet$ separates $\overline{\mathcal{S}_k^\bullet}$ from $\overline{\mathcal{S}_{k+1}^\bullet}$.

- $\mathrm{T}_2^\bullet$ separates $\overline{\mathcal{C}_k^\bullet}$ from $\overline{\mathcal{S}_k^\bullet}$ since any graph in $\overline{\mathcal{C}_k^\bullet}$ has treewidth at most two. $\mathrm{CC}_k^\bullet$ separates $\overline{\mathcal{S}_k^\bullet}$ from $\overline{\mathcal{C}_k^\bullet}$ as no graph in $\overline{\mathcal{S}_k^\bullet}$ has chordless cycle of length $k$.

- $\mathrm{CC}_3^\bullet$ separates $\overline{\mathcal{K}_k^\bullet}$ from $\overline{\mathcal{S}_k^\bullet}$ since no graph in $\overline{\mathcal{K}_k^\bullet}$ has chordless cycle of length more than three. $\mathrm{T}_k^\bullet$ separates $\overline{\mathcal{S}_k^\bullet}$ from $\overline{\mathcal{K}_k^\bullet}$ since every graph in $\overline{\mathcal{K}_k^\bullet}$ has the treewidth of at most $k$.

- The above proof works for separating $\overline{\mathcal{C}_k^\bullet}$ and $\overline{\mathcal{K}_k^\bullet}$.

□

Figure 3: Hierarchy of GNN models. Only showing WL variants, $C_k$-DHN, and $K_k$-DHN. Models without (transitive) arrows means they are not comparable, e.g., 2-WL and DHN with $\leqslant 4$ cliques are incomparable.

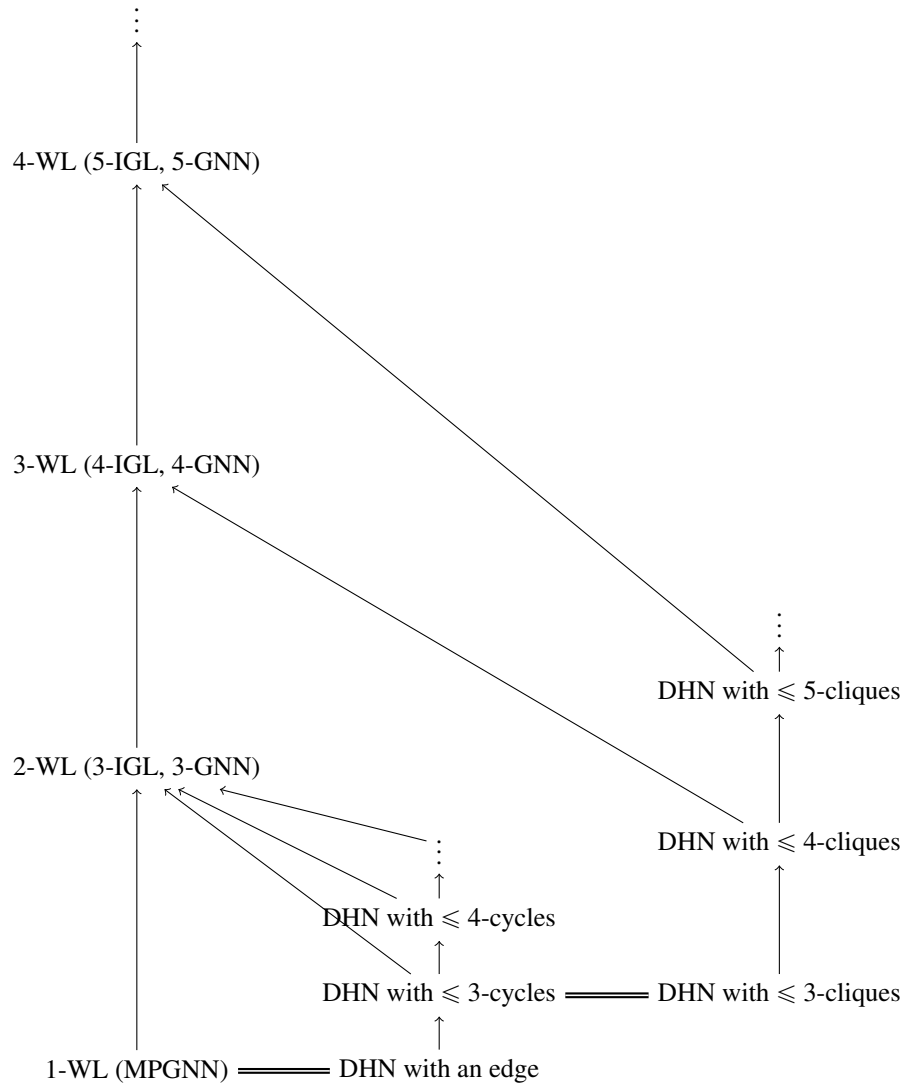
